# Provable and Efficient Dataset Distillation for Kernel Ridge Regression

**Yilan Chen**
UCSD CSE
yic031@ucsd.edu

**Wei Huang**
RIEKN AIP
wei.huang.vr@riken.jp

**Tsui-Wei Weng**
UCSD HDSI
lweng@ucsd.edu

## Abstract

Deep learning models are now trained on increasingly larger datasets, making it crucial to reduce computational costs and improve data quality. Dataset distillation aims to distill a large dataset into a small synthesized dataset such that models trained on it can achieve similar performance to those trained on the original dataset. While there have been many empirical efforts to improve dataset distillation algorithms, a thorough theoretical analysis and provable, efficient algorithms are still lacking. In this paper, by focusing on dataset distillation for kernel ridge regression (KRR), we show that one data point per class is already necessary and sufficient to recover the original model's performance in many settings. For linear ridge regression and KRR with surjective feature mappings, we provide necessary and sufficient conditions for the distilled dataset to recover the original model's parameters. For KRR with injective feature mappings of deep neural networks, we show that while one data point per class is not sufficient in general, $k+1$ data points can be sufficient for deep linear neural networks, where $k$ is the number of classes. Our theoretical results enable directly constructing analytical solutions for distilled datasets, resulting in a provable and efficient dataset distillation algorithm for KRR. We verify our theory experimentally and show that our algorithm outperforms previous work such as KIP while being significantly more efficient, e.g. $15840\times$ faster on CIFAR-100. Our code is available at GitHub.

## 1 Introduction

Deep learning models are now trained on increasingly massive datasets, incurring substantial computational costs and data quality challenges. For instance, Llama 3 was pre-trained on over 15 trillion tokens, while the training of GPT-4 exceeded $100 million. Reducing these burdens is crucial. Dataset distillation [34] aims to distill a large dataset into a small synthesized dataset such that models trained on it can achieve similar performance to those trained on the original dataset. A good small distilled dataset is not only useful in saving computational cost and improving data quality but also has various applications such as continual learning [39, 40, 35], privacy protection [24, 40, 18, 5, 1], and neural architecture search [31, 39].

While there have been many empirical efforts to improve dataset distillation algorithms [24, 39, 2, 38], a thorough theoretical analysis is still lacking. Izzo and Zou [9] show single distill data is sufficient for a class of generalized linear models with one-dimensional output, where the data is assumed to follow a generalized exponential density function and the negative log-likelihood is optimized by gradient descent. For a linear ridge regression (LRR), Izzo and Zou [9] show $d$ data points are needed to recover the original model's parameter for all regularization at the same time, where $d$ is the dimension of the data and is large even for small datasets like MNIST [13] and CIFAR [12] ($d = 784$ and $3072$) in computer vision. For kernel regression with Gaussian kernel, they show $n$ data points are necessary, where $n$ is the number of original data points and can be large for modern datasets, e.g.

Table 1: Comparison with existing theoretical analysis of dataset distillation. The number of distilled data needed to recover original model's performance and models analyzed. "-" means not applicable. For linear ridge regression (LRR) and kernel ridge regression (KRR) with subjective feature mapping, our results only need one distilled data per class ($k \leq d$ in our setting), which is far less than the existing work [9, 21] that require $n$ or $p$ points. As an example, $k = 10, d = 3072, n = 50000$ for CIFAR-10. The $k, d, n$ of standard datasets are listed in Table 2. $p$ is the dimension of feature mapping $\phi : \mathbb{R}^d \mapsto \mathbb{R}^p$.

| | LRR | Kernel ridge regression (KRR) | |
|---|---|---|---|
| | | surjective $\phi$ | non-surjective $\phi$ |
| Izzo and Zou [9] | $d$ | - | $n$ (Gaussian Kernel) |
| Maalouf et al. [21] | - | - | $p$ (Shift-invariant Kernels) |
| **Our work** | $k, (k \leq d)$ | $k, (k \leq p)$ (Invertible NNs, FCNN, CNN, Random Fourier Features) | $p$ in general (Deep nonlinear NNs). $k + 1$ for deep linear NNs |

$n = 60000$ and $50000$ for MNIST and CIFAR (see Table 2). Maalouf et al. [21] use Random Fourier Features (RFF) to approximate shif-invariant kernels that may have an infinite-dimensional feature space, and construct $p$ distilled data for such RFF model, where $p$ is the dimension of the RFF model and can be $\Omega(\sqrt{n} \log n)$ in general cases. The results in [9, 21], however, have a large gap compared with the empirical evidence that one data point per class can often achieve comparable performance to the original model [24, 39, 2, 38].

In this paper, by focusing on dataset distillation for kernel ridge regression (KRR), we show that *one data point per class is already necessary and sufficient to recover the original model's performance in many settings*, which is far less than $n$ or $p$ data points needed in prior works [9, 21]. Besides, our analysis is more general than prior works [9, 21] and can handle more and different models, including invertible neural networks, fully-connected neural networks (FCNN), Convolutional neural networks (CNN), and Random Fourier Features (RFF). Table 1 compares our theoretical results with previous analysis. We summarize our contributions as follows.

- In Sec. 4.1 and 5, for linear ridge regression (LRR) and KRR with surjective feature mappings, we show that one distilled data point per class is *necessary and sufficient* to recover the original model's parameters and provide necessary and sufficient conditions for such distilled datasets. In addition, we show how to find distilled data that is close to real data in Sec. 4.2.

- In Sec. 5.2, for KRR with injective feature mappings of deep neural networks (NNs), we show that one data point per class is in general *not sufficient* to recover the original model's parameters. However, $k + 1$ data points can be sufficient for deep linear NNs, where $k$ is the number of classes.

- Our theoretical results enable us to directly construct analytical solutions for the distilled datasets, resulting in a provable and efficient dataset distillation algorithm for KRR in Algorithm 1. We verify our theory experimentally and show that our algorithm outperforms previous SOTA dataset distillation algorithm KIP [25] while being significantly more efficient, e.g. $15840\times$ faster on CIFAR-100.

- In Sec.6, we show our theoretical results can be used for several applications. First, it can be used as necessary or sufficient conditions for KIP-type algorithms to converge to a global minimum even if the loss function is highly non-convex. Second, our distilled dataset for KRR can provably preserve the privacy of the original dataset while having a performance guarantee.

## 2 Related works

**Dataset distillation.** Dataset distillation aims to distill a large dataset into a small synthesized dataset such that models trained on it can achieve similar performance to those trained on the original dataset. Previous approaches can be mainly divided into four categories [29]: 1) Meta-model Matching: this category formulates the problem as a bilevel optimization problem and maximize the performance of the model trained on the distilled dataset [34]. Some recent works such as KIP [24, 25], FRePo [40], RFAD [17], and RCIG [18] approximate the inner loop optimization of training neural networks by

KRR with Neural Tangent Kernel [10] or neural network Gaussian process (NNGP) kernels [14]. 2) Gradient Matching: this category minimizes the distance between the gradients of models trained on the original dataset and distilled dataset [39, 37, 15, 11]. 3) Trajectory Matching: this category aims to match the training trajectories of models trained on the original dataset and distilled dataset [2, 4, 8, 7]. 4) Distribution Matching: this approach directly matches the distribution of the original dataset and distilled dataset via a single-level optimization [38, 33, 36]. Our work is closely related to kernel-based dataset distillation algorithms [24, 25, 40, 17, 18] in category (1). Our theoretical analysis provides theoretical foundations and implications for these kernel-based algorithms.

**Theoretical analysis of dataset distillation.** In addition to the papers discussed in the introduction, Maalouf et al. [19] propose an efficient algorithm to construct a $d^2 + 1$ core set of the original dataset for least mean squares problems. Maalouf et al. [20] further propose to use the SVD of the original dataset to construct a distilled dataset of size $d$. Tukan et al. [32] utilize the idea of subset selection to improve the initialization and training procedure of dataset distillation. Our paper focuses on KRR and constructs $k$ distilled data analytically, where $k$ is usually much less than $d$ (see Table 2).

## 3   Preliminaries

### 3.1   Dataset Distillation

For an original dataset $\{\boldsymbol{x}_i, \boldsymbol{y}_i\}_{i=1}^n$, we denote $\mathbf{X} = [\boldsymbol{x}_1, \ldots, \boldsymbol{x}_n] \in \mathbb{R}^{d \times n}$ and $\mathbf{Y} = [\boldsymbol{y}_1, \ldots, \boldsymbol{y}_n] \in \mathbb{R}^{k \times n}$, where $d$ is the dimension of the data, $k$ is the dimension of the label or the number of the classes, and $n$ is the number of data points. The goal of dataset distillation is to create a synthetic dataset $\mathbf{X}_S = [\boldsymbol{x}_{S_1}, \ldots, \boldsymbol{x}_{S_m}] \in \mathbb{R}^{d \times m}$ and $\mathbf{Y}_S = [\boldsymbol{y}_{S_1}, \ldots, \boldsymbol{y}_{S_m}] \in \mathbb{R}^{k \times m}$, with the number of distilled data points $m \ll n$, such that a model trained on this synthetic dataset $(\mathbf{X}_S, \mathbf{Y}_S)$ can achieve similar performance to those trained on the original dataset.

Table 2: $k$ (number of class), $d$ (dimension of data), and $n$ (number of training data) of standard datasets.

| Dataset | $k$ | $d$ | $n$ |
|---|---|---|---|
| MNIST [13] | 10 | 784 | 60000 |
| CIFAR-10 [12] | 10 | 3072 | 50000 |
| CIFAR-100 [12] | 100 | 3072 | 50000 |
| ImageNet-1k [28] | 1000 | 196608 | 1281167 |

As the data dimension is usually larger than the label dimension in practice, e.g. MNIST has $d = 728, k = 10$ and other datasets have even larger $d$, we consider $d \geq k$ in this paper. For a matrix $\mathbf{A}$, we use $\mathbf{A}^+$ to denote its pseudoinverse and $\mathrm{Range}\,(\mathbf{A})$ to denote its range space.

### 3.2   Dataset Distillation for Kernel Ridge Regression (KRR)

**Original model**: Given a kernel $K(\boldsymbol{x}, \boldsymbol{x}') = \langle \phi(\boldsymbol{x}), \phi(\boldsymbol{x}') \rangle$, where $\phi : \mathbb{R}^d \mapsto \mathbb{R}^p$ is the feature mapping from input space to a feature space of dimension $p$, a KRR model $f(\boldsymbol{x}) = \mathbf{W}\phi(\boldsymbol{x})$ trained on original data set with a predefined regularization $\lambda \geq 0$ tries to minimize following objective

$$\min_{\mathbf{W}} \|\mathbf{W}\phi(\mathbf{X}) - \mathbf{Y}\|_F^2 + \lambda \|\mathbf{W}\|_F^2$$

where $\mathbf{W} \in \mathbb{R}^{k \times p}$ and $\phi(\mathbf{X}) = [\phi(\boldsymbol{x}_1), \ldots, \phi(\boldsymbol{x}_n)] \in \mathbb{R}^{p \times n}$. The solution can be computed analytically as $\mathbf{W} = \mathbf{Y}\phi_\lambda(\mathbf{X})^+$, where

$$\phi_\lambda(\mathbf{X})^+ = \begin{cases} \left(K(\mathbf{X}, \mathbf{X}) + \lambda \mathbf{I}_n\right)^{-1} \phi(\mathbf{X})^\top = \phi(\mathbf{X})^\top \left(\phi(\mathbf{X})\phi(\mathbf{X})^\top + \lambda \mathbf{I}_p\right)^{-1}, & \text{if } \lambda > 0, \\ \phi(\mathbf{X})^+, & \text{if } \lambda = 0. \end{cases}$$

and $K(\mathbf{X}, \mathbf{X}) = \phi(\mathbf{X})^\top \phi(\mathbf{X}) \in \mathbb{R}^{n \times n}$. $\phi_\lambda(\mathbf{X})$ can be considered as regularized features. Linear ridge regression is a special case of kernel ridge regression (KRR) with $\phi(\boldsymbol{x}) = \boldsymbol{x}$.

KRR is used in many dataset distillation algorithms [24, 25, 40, 17, 18]. In this paper, we mainly consider a finite-dimensional $\phi$. This matches the practical neural networks which are usually used in dataset distillation. For shift-invariant kernels with infinite-dimensional RKHS space, e.g. Gaussian kernel, they can be well approximated by finite-dimensional random Fourier features [27, 16].

**Distilled dataset model**: Similarly, a KRR trained on distilled dataset with regularization $\lambda_S \geq 0$ is $f_S(\boldsymbol{x}) = \mathbf{W}_S \phi(\boldsymbol{x})$, where $\mathbf{W}_S = \mathbf{Y}_S \phi_{\lambda_S}(\mathbf{X}_S)^+ \in \mathbb{R}^{k \times d}$. Additionally, denote $\mathbf{X}_{\lambda_S} = \phi_{\lambda_S}(\mathbf{X}_S)$ with $\phi(\boldsymbol{x}) = \boldsymbol{x}$, i.e.

$$\mathbf{X}_{\lambda_S}^+ = \begin{cases} \left(\mathbf{X}_S^\top \mathbf{X}_S + \lambda_S \mathbf{I}_m\right)^{-1} \mathbf{X}_S^\top = \mathbf{X}_S^\top \left(\mathbf{X}_S \mathbf{X}_S^\top + \lambda_S \mathbf{I}_d\right)^{-1}, & \text{if } \lambda_S > 0, \\ \mathbf{X}_S^+, & \text{if } \lambda_S = 0. \end{cases}$$

The goal of dataset distillation here is to find $(\mathbf{X}_S, \mathbf{Y}_S)$ such that $\mathbf{W}_S = \mathbf{W}$, where $\mathbf{W}$ is supposed to be given, i.e. can be computed from the original dataset $(\mathbf{X}, \mathbf{Y})$.

# 4 Dataset Distillation for Linear Ridge Regression (LRR)

In this section, we first analyze the dataset distillation for the linear ridge regression (LRR). In Sec. 4.1, for a LRR model, we show that $k$ distilled data points (one per class) are necessary and sufficient to guarantee $\mathbf{W}_S = \mathbf{W}$. We provide analytical solutions of such $\mathbf{X}_S$ allowing us to compute the distilled dataset analytically instead of having to learn it heuristically in existing works [24, 25, 40, 17, 18]. Then, in Sec 4.2, we show how to find distilled data that is close to real data. Lastly, for fixed data and only distilling labels, we show $d$ points are needed in Sec 4.3.

## 4.1 Analytical Computation for Linear Ridge Regression

**Theorem 4.1.** *When $m < k$, there is no $\mathbf{X}_S$ can guarantee $\mathbf{W}_S = \mathbf{W}$ unless the columns of $\mathbf{W}$ are in the range space of $\mathbf{Y}_S$. When $m \geq k$ and $\mathbf{Y}_S$ is rank $k$, let $r = \min(m, d)$ and take $\mathbf{D} = \mathbf{Y}_S^+ \mathbf{W} + \left(\mathbf{I}_m - \mathbf{Y}_S^+ \mathbf{Y}_S\right)\mathbf{Z}$, where $\mathbf{Z} \in \mathbb{R}^{m \times d}$ is any matrix of the same size as $\mathbf{X}_S^\top$. Suppose the reduced SVD of $\mathbf{D}$ is $\mathbf{D} = \mathbf{V} diag(\sigma_1', \dots, \sigma_r')\mathbf{U}^\top$ with $\sigma_1' \geq \cdots \geq \sigma_r' \geq 0$, the following results hold:*

*1. $\lambda_S > 0$: $\mathbf{W}_S = \mathbf{W}$ if and only if, for any $\mathbf{D}$ defined above, $\lambda_S \leq \frac{1}{4\sigma_1'^2}$ and $\mathbf{X}_S = $*

$$\mathbf{U} diag(\sigma_1, \dots, \sigma_r)\mathbf{V}^\top \text{ where } \sigma_i = \begin{cases} 0, & \text{if } \sigma_i' = 0, \\ \frac{1 \pm \sqrt{1 - 4\lambda_S \sigma_i'^2}}{2\sigma_i'}, & \text{otherwise.} \end{cases}$$

*2. $\lambda_S = 0$: $\mathbf{W}_S = \mathbf{W}$ if and only if $\mathbf{X}_S = \mathbf{D}^+$ for any $\mathbf{D}$ defined above.*

*Proof sketch.* The proof can be found in Appendix C. The key idea is that we want to solve $\mathbf{X}_S$ from $\mathbf{W}_S = \mathbf{Y}_S \mathbf{X}_{\lambda_S}^+ = \mathbf{W}$ (Note: $\mathbf{W}$ is given and we can select/decide $\mathbf{Y}_S$). When $m < k$, this is an overdetermined system for $\mathbf{X}_{\lambda_S}^+$. There is no solution for $\mathbf{X}_{\lambda_S}^+$ in general therefore no solution for $\mathbf{X}_S$. When $m \geq k$ and $\mathbf{Y}_S$ is rank $k$, the solutions of $\mathbf{X}_{\lambda_S}^+$ are given by $\mathbf{X}_{\lambda_S}^+ = \mathbf{Y}_S^+ \mathbf{W} + \left(\mathbf{I}_m - \mathbf{Y}_S^+ \mathbf{Y}_S\right)\mathbf{Z}$, where $\mathbf{Z} \in \mathbb{R}^{m \times d}$ is any matrix of the same size as $\mathbf{X}_{\lambda_S}^+$. However, not all such $\mathbf{X}_{\lambda_S}^+$ corresponds to a $\mathbf{X}_S$. To solve $\mathbf{X}_S$, we assume we have the SVD of $\mathbf{X}_S$ and solve it from the SVD of $\mathbf{X}_{\lambda_S}^+$. $\square$

Intuitively, original dataset $(\mathbf{X}, \mathbf{Y})$ is compressed into $\mathbf{X}_S$ through original model's parameter $\mathbf{W} = \mathbf{Y}\left(\mathbf{X}^\top \mathbf{X} + \lambda \mathbf{I}_n\right)^{-1}\mathbf{X}^\top$. When $m = k$, i.e. one distilled data per class, $\mathbf{D} = \mathbf{Y}_S^+ \mathbf{W}$ is deterministic and $\mathbf{X}_S$ is fully determined by $\mathbf{W}$ and $\mathbf{Y}_S$. In this case, when $\lambda_S = 0$ and $\mathbf{W}$ is full rank, $\mathbf{X}_S$ can be easily computed as $\mathbf{X}_S = \mathbf{W}^+ \mathbf{Y}_S$. As an example, Figure 1 shows the distilled data for MNIST and CIFAR-100 when $m = 10/100$. When $m > k$, i.e. more than one distilled data per class, there exist infinitely many distilled datasets since $\mathbf{Z}$ is a free variable to choose. When $m = n$, one can verify that $\mathbf{X}$ is a distilled dataset for itself by taking $\mathbf{Y}_S = \mathbf{Y}$ and $\mathbf{Z} = \left(\mathbf{X}^\top \mathbf{X} + \lambda \mathbf{I}_n\right)^{-1}\mathbf{X}^\top$. When $m > n$, we can generate more data than original dataset. Compared with [9] that needs $d$ data, our approach only requires $k$ data and usually $k \leq d$ in practice. Our result is also more flexible to distill any $m \geq k$ data points.

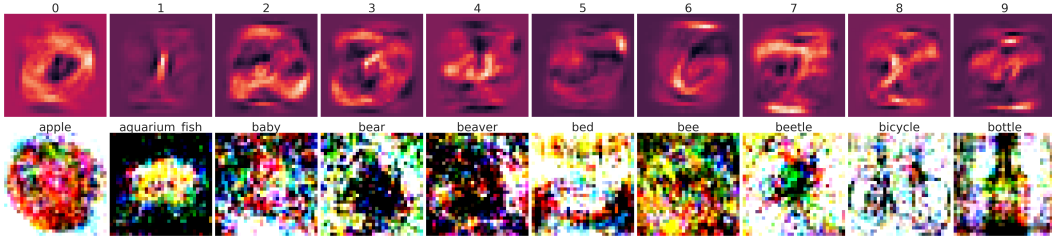

Figure 1: Distilled data of MNIST (first row) and CIFAR-100 (second row) for LRR when $m = k$.

**Discussion.** The requirement for $\lambda_S$ can be easily satisfied by setting $\lambda_S \leq \frac{1}{4\sigma_1'^2}$ for a given $\mathbf{D}$. If we want to fix a predefined $\lambda_S$, e.g. $\lambda_S = \lambda$, we need to sample different $\mathbf{D}$ (by sampling different $\mathbf{Z}$) so that $\lambda_S \leq \frac{1}{4\sigma_1'^2}$ is satisfied. Theorem 4.1 generally suggests that a smaller $\lambda_S$ is better for

constructing distilled data. Practical algorithms, e.g. KIP, FRePo, and RFAD, usually use a very small regularization, which may already satisfy the requirement.

In practice, we usually do not want the dataset to be singular. Below we show that it is easy to satisfy as long as $\mathbf{Z}$ is full rank and the rows of $\mathbf{W}$ and $\mathbf{Z}$ are linearly independent.

**Proposition 4.1.** *When $m \geq k$ and $\mathbf{Y}_S, \mathbf{W}$ are rank $k$, the $\mathbf{X}_S$ in Theorem 4.1 is full rank for any full-rank $\mathbf{Z}$ such that* $\mathrm{Range}\left(\mathbf{W}^\top\right) \cap \mathrm{Range}\left(\mathbf{Z}^\top\right) = \{\mathbf{0}\}$.

## 4.2 Finding Realistic Distilled Data

In Theorem 4.1, any $\mathbf{D}$ satisfying the condition will guarantee the distilled dataset model to recover the original model's performance. For example, we can choose $\mathbf{Z}$ to be a random Gaussian matrix. However, to make the distilled data more realistic and generalizable, we can select $m$ real training data $\hat{\mathbf{X}}_S$ as initialization of distilled data and find the distilled data that is closest to $\hat{\mathbf{X}}_S$.

**Corollary 4.1.1.** *Given fixed $\hat{\mathbf{X}}_S, \lambda_S,$ and $\mathbf{Y}_S,$ the $\mathbf{D}$ that satisfies Theorem 4.1 and minimize $\left\| \mathbf{D} - \hat{\mathbf{X}}_{\lambda_S}^+ \right\|_F$ is*

$$\mathbf{D} = \mathbf{Y}_S^+ \mathbf{W} + \left(\mathbf{I}_m - \mathbf{Y}_S^+ \mathbf{Y}_S\right)\left(\hat{\mathbf{X}}_{\lambda_S}^+ - \mathbf{Y}_S^+ \mathbf{W}\right),$$

*where $\hat{\mathbf{X}}_{\lambda_S}^+$ is defined analogous to $\mathbf{X}_{\lambda_S}^+$. Taking $\mathbf{Y}_S = \mathbf{W}\hat{\mathbf{X}}_{\lambda_S}$ can further minimize the distance.*

When $\lambda_S = 0$, $\mathbf{Y}_S = \mathbf{W}\hat{\mathbf{X}}_S$ is the prediction of original model for $\hat{\mathbf{X}}_S$. Combining this with Theorem 4.1, we summarize the computation of the distilled data in Algorithm 1 with $\phi(\boldsymbol{x}) = \boldsymbol{x}$. Figure 2 shows some distilled data that is close to the real data or generated with random noise.

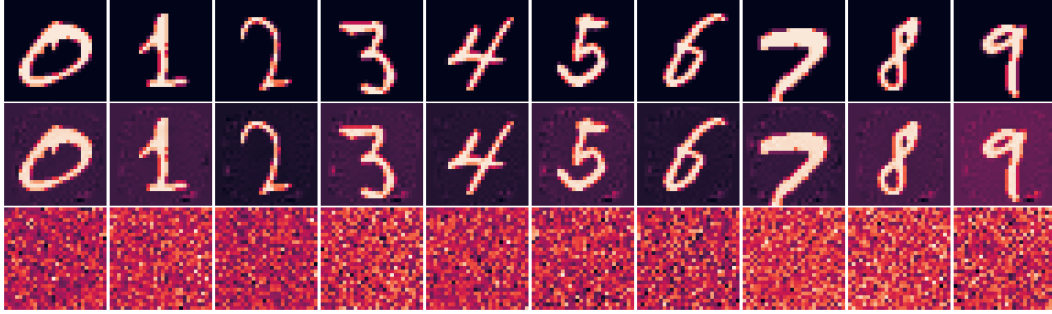

(a) MNIST with IPC=50.

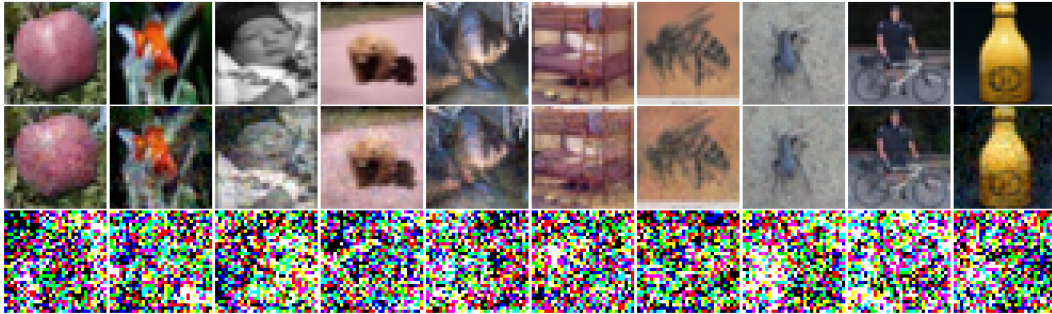

(b) CIFAR-100 with IPC=5.

Figure 2: Initialized data (first row), distilled data generated from real images using techniques in Sec 4.2 (second row), and distilled data generated from random noise using techniques in Sec 4.1 (third row) for a LRR with $m = 500$ on MNIST and CIFAR-100. IPC: images per class.

## 4.3 Label Distillation

If we fix the $\mathbf{X}_S$ and only distill labels, as also shown in [24], we need at least $m = d$ to guarantee $\mathbf{W}_S = \mathbf{W}$ because labels have fewer learnable parameters than data.

**Theorem 4.2.** *For any fixed $\mathbf{X}_S,$*

1. *when $m < d$, there is no $\mathbf{Y}_S$ can guarantee $\mathbf{W}_S = \mathbf{W}$ in general unless the rows of $\mathbf{W}$ are in the row space of $\mathbf{X}_{\lambda_S}^+$. The least square solution is $\mathbf{Y}_S = \mathbf{W}\mathbf{X}_{\lambda_S}$ and $\|\mathbf{W}_S - \mathbf{W}\| = \left\|\mathbf{W}\left(\mathbf{X}_{\lambda_S}\mathbf{X}_{\lambda_S}^+ - \mathbf{I}_d\right)\right\|$.*

2. *when $m \geq d$, if $\mathbf{X}_S$ is rank $d$, then $\mathbf{Y}_S = \mathbf{W}\mathbf{X}_{\lambda_S}$ is sufficient for $\mathbf{W}_S = \mathbf{W}$.*

# 5 Dataset Distillation for Kernel Ridge Regression (KRR)

In the last section, we analyzed the dataset distillation for LRR. However, more complex models such as KRR and neural networks (NNs) are usually used in practice for better performance. Therefore, it is crucial to extend the analysis to these practical settings. In this section, we first extend the results of LRR to KRR in the feature space and then construct distilled data from desired features by considering two cases – subjective and non-surjective feature mappings.

The results in Theorem 4.1 of LRR can be directly extended to the KRR in the feature space by replacing data $\mathbf{X}_S$ with features $\phi(\mathbf{X}_S)$ in Theorem 4.1. For completeness, we state it below.

**Theorem 5.1.** *When $m < k$, there is no $\phi(\mathbf{X}_S)$ can guarantee $\mathbf{W}_S = \mathbf{W}$ unless the columns of $\mathbf{W}$ are in the range space of $\mathbf{Y}_S$. When $m \geq k$ and $\mathbf{Y}_S$ is rank $k$, let $r = \min(m, p)$ and take $\mathbf{D} = \mathbf{Y}_S^+\mathbf{W} + \left(\mathbf{I}_m - \mathbf{Y}_S^+\mathbf{Y}_S\right)\mathbf{Z}$, where $\mathbf{Z} \in \mathbb{R}^{m \times p}$ is any matrix of the same size as $\phi(\mathbf{X}_S)^\top$. Suppose the reduced SVD of $\mathbf{D}$ is $\mathbf{D} = \mathbf{V}diag(\sigma_1', \ldots, \sigma_r')\mathbf{U}^\top$ with $\sigma_1' \geq \cdots \geq \sigma_r' \geq 0$, the following results hold:*

1. *$\lambda_S > 0$: $\mathbf{W}_S = \mathbf{W}$ if and only if, for any $\mathbf{D}$ defined above, $\lambda_S \leq \frac{1}{4\sigma_1'^2}$ and $\phi(\mathbf{X}_S) = \mathbf{U}diag(\sigma_1, \ldots, \sigma_r)\mathbf{V}^\top$ where $\sigma_i = \begin{cases} 0, & \text{if } \sigma_i' = 0, \\ \frac{1 \pm \sqrt{1 - 4\lambda_S\sigma_i'^2}}{2\sigma_i'}, & \text{otherwise.} \end{cases}$*

2. *$\lambda_S = 0$: $\mathbf{W}_S = \mathbf{W}$ if and only if $\phi(\mathbf{X}_S) = \mathbf{D}^+$ for any $\mathbf{D}$ defined above.*

This shows that in the feature space, $k$ features are necessary and sufficient to recover the original model's parameter. However, what we get is the feature of distilled data $\phi(\mathbf{X}_S)$ instead of distilled data $\mathbf{X}_S$ itself. To get $\mathbf{X}_S$, we need to construct the data from the features. To do this, we consider two cases – surjective and non-surjective $\phi$. For subjective $\phi$, we show that we can directly construct $\mathbf{X}_S$ from $\phi(\mathbf{X}_S)$. For non-surjective $\phi$ such as neural networks (NNs), we show one data per class is in general not sufficient, but $k + 1$ data points can be sufficient for deep linear neural networks.

## 5.1 Surjective Feature Mapping

When $\phi$ is surjective or bijective, we can always find a $\mathbf{X}_S$ for a desired $\phi(\mathbf{X}_S)$. In this case, $k$ distilled data (one data per class) is sufficient to recover the original model's performance, in contrast to [21] that needs $p$ distilled data. We summarize the computation of the distilled data in Algorithm 1. Here we give some examples of surjective/bijective $\phi$.

**Example 5.1** (Invertible NN)**.** If $\phi$ is invertible such as invertible NNs used in normalizing flows, then we can directly compute $\boldsymbol{x} = \phi^{-1}(\phi(\boldsymbol{x}))$.

**Example 5.2** (Fully-connected NN (FCNN))**.** For a $(L + 1)$-layer FCNN $f(x) = \mathbf{W}\phi(x)$ and

$$\phi(\boldsymbol{x}) = \sigma(\mathbf{W}^{(L)}\sigma(\cdots \mathbf{W}^{(2)}\sigma(\mathbf{W}^{(1)}\boldsymbol{x}))).$$

where $\sigma$ is a surjective or bijective activation function such as LeakyReLU, and $\mathbf{W}^{(l)} \in \mathbb{R}^{d_l \times d_{l-1}}$ with $d = d_0 \geq d_1 \geq \cdots \geq d_L$ for $l \in [L]$. If all $\mathbf{W}^{(l)}$ are full rank, given $\phi(x)$, we can compute

$$\boldsymbol{x} = \left(\mathbf{W}^{(1)}\right)^+ \sigma^{-1}\left(\cdots \left(\mathbf{W}^{(L)}\right)^+ \sigma^{-1}(\phi(\boldsymbol{x}))\right),$$

where $\sigma^{-1}$ is any right inverse of $\sigma$. When some $\mathbf{W}^{(l)}$ are not full rank, we can still compute an approximated solution.

**Example 5.3** (Convolutional Neural Network (CNN))**.** CNN is known as a special type of FCNN. To illustrate, we give an example of a convolution layer of $2 \times 2$ filter. Let $\boldsymbol{w} \in \mathbb{R}^4$ be a convolutional

filter of size 2. Then the convolution operation with stride 1 can be represented as

$$\phi(\boldsymbol{x}) = \begin{bmatrix} \boldsymbol{w}_1 & \boldsymbol{w}_2 & 0 & 0 & \cdots & 0 & \boldsymbol{w}_3 & \boldsymbol{w}_4 & 0 & \cdots & 0 \\ 0 & \boldsymbol{w}_1 & \boldsymbol{w}_2 & 0 & \cdots & 0 & 0 & \boldsymbol{w}_3 & \boldsymbol{w}_4 & \cdots & 0 \\ \vdots & \vdots & \vdots & \ddots & \vdots & \vdots & \vdots & \ddots & \vdots & \vdots & \vdots \\ 0 & 0 & 0 & \cdots & \boldsymbol{w}_1 & \boldsymbol{w}_2 & 0 & \cdots & 0 & \boldsymbol{w}_3 & \boldsymbol{w}_4 \end{bmatrix} \boldsymbol{x}$$

If the data is three-channel images, same operation can be done for each channel followed by a matrix that sums over three channels. When the matrix equation has a solution, the data can be solved from the feature.

**Example 5.4** (Random Fourier Features (RFF) [27]). A shift-invariant kernel $K(\boldsymbol{x}, \boldsymbol{x}') = K(\boldsymbol{x} - \boldsymbol{x}')$ can be approximated by random Fourier features $K(\boldsymbol{x}, \boldsymbol{x}') \approx \langle \phi(\boldsymbol{x}), \phi(\boldsymbol{x}') \rangle$. Here $\phi(\boldsymbol{x}) = \sqrt{\frac{2}{p}} \left[ \cos(\boldsymbol{a}_1^\top \boldsymbol{x} + b_1), \ldots, \cos(\boldsymbol{a}_p^\top \boldsymbol{x} + b_p) \right]^\top \in \mathbb{R}^p$ where $\boldsymbol{a}_1, \ldots, \boldsymbol{a}_p \in \mathbb{R}^d$ are independent samples from a distribution $\mathbb{P}(\boldsymbol{a}) = \frac{1}{2\pi} \int e^{-j\boldsymbol{a}^\top \Delta} K(\Delta) \, d\Delta$ (Fourier transform of $K(\Delta)$) and $b_1, \ldots, b_p$ are i.i.d. sampled from the uniform distribution on $[0, 2\pi]$. For example, Gaussian kernel $K(\boldsymbol{x}, \boldsymbol{x}') = e^{-\frac{\|\boldsymbol{x} - \boldsymbol{x}'\|_2^2}{2\sigma}}$ has $\mathbb{P}(\boldsymbol{a}) = \mathcal{N}(0, \sigma^{-2} \mathbf{I}_d)$. Denote $\mathbf{A} = [\boldsymbol{a}_1, \ldots, \boldsymbol{a}_p]^\top$ and $\boldsymbol{b} = [b_1, \ldots, b_p]^\top$, then we have $\phi(\boldsymbol{x}) = \sqrt{\frac{2}{p}} \cos(\mathbf{A}\boldsymbol{x} + \boldsymbol{b})$. Whenever $p \leq d$ and $\mathbf{A}$ is rank $p$, given $\phi(\boldsymbol{x})$ we can solve $\boldsymbol{x}$ as,

$$\boldsymbol{x} = \mathbf{A}^+ \left( \arccos \sqrt{\frac{p}{2}} \phi(\boldsymbol{x}) - \boldsymbol{b} \right).$$

To ensure the computed $\sqrt{\frac{p}{2}} \phi(\mathbf{X}_S) \in [-1, 1]$, we can normalize it by its largest absolute value, which is equivalently scaling $\mathbf{D}$ in Theorem 5.1 and does not affect the direction of $\mathbf{W}_S$.

[21] use random Fourier features to approximate shif-invariant kernels that may have an infinite-dimensional feature space, and construct $p$ distilled data for such RFF model, where $p \in \Omega(\sqrt{n} \log n)$ in general cases. Their construction, however, only uses label distillation and the $\mathbf{X}_S$ can be any random data. Our analysis constructs $\mathbf{X}_S$ explicitly and shows that whenever the dimension of RFF $p$ needs to approximate shif-invariant kernels is less than $d$, $k$ distilled data suffice to recover the performance of the original RFF model and approximate the original KRR with shif-invariant kernels.

## 5.2 Non-surjective Feature Mapping

When $\phi$ is injective or non-surjective, given a $\phi(\mathbf{X}_S)$, we may not find an exactly matched $\mathbf{X}_S$. However, we can find an approximated distilled data $\hat{\mathbf{X}}_S$ first and then adjust $\mathbf{Y}_S$ to ensure $\mathbf{W}_S \approx \mathbf{W}$. As in the label distillation for LRR case, $m \geq p$ distilled labels can guarantee $\mathbf{W}_S = \mathbf{W}$.

Below we show one data per class is in general not sufficient for non-surjective $\phi$, but $k + 1$ can be sufficient for deep linear NNs. Consider a deep NN, $f(x) = \mathbf{W}\phi(x)$ with

$$\phi(\boldsymbol{x}) = \sigma(\mathbf{W}^{(L)} \sigma(\cdots \mathbf{W}^{(2)} \sigma(\mathbf{W}^{(1)} \boldsymbol{x}))).$$

where $\sigma$ is an invertible activation function such as LeakyReLU, Sigmoid, and $\mathbf{W}^{(l)} \in \mathbb{R}^{d_l \times d_{l-1}}$ with $d = d_0 < d_1 = \cdots = d_L = p$ for $l \in [L]$. $\phi(\boldsymbol{x})$ is an injective function in this definition.

**Theorem 5.2.** *For a deep nonlinear NN defined above with fixed $\phi$, assume $\mathbf{W}^{(2)}, \ldots, \mathbf{W}^{(L)}$ are full rank. Suppose $\lambda_S = 0$ and $\mathbf{Y}_S$ is rank $k$. When $m = k$, there is no distilled data $\mathbf{X}_S$ that can guarantee $\mathbf{W}_S = \mathbf{W}$ in general useless the columns of $\sigma^{-1} \left( \left(\mathbf{W}^{(2)}\right)^{-1} \cdots \left(\mathbf{W}^{(L)}\right)^{-1} \sigma^{-1} \left((\mathbf{Y}_S^+ \mathbf{W})^+\right) \right)$ are in the range space of $\mathbf{W}^{(1)}$.*

When $m = k$, only $\phi(\mathbf{X}_S) = (\mathbf{Y}_S^+ \mathbf{W})^+$ that can guarantee $\mathbf{W}_S = \mathbf{W}$. One data per class is not sufficient in general as long as $(\mathbf{Y}_S^+ \mathbf{W})^+$ is not in the range space of $\phi$. Although $k$ data is not sufficient, we show $k + 1$ data can be sufficient for deep linear neural networks, where $\sigma(\boldsymbol{x}) = \boldsymbol{x}$.

**Theorem 5.3.** *For a deep linear NN defined above with fixed $\phi$, assume $\mathbf{W}^{(2)}, \ldots, \mathbf{W}^{(L)}$ are full rank. Suppose $\lambda_S = 0$ and $\mathbf{Y}_S, \mathbf{W}$ are rank $k$. Denote $\mathbf{H} = \left[ \prod_{l=1}^{L} \mathbf{W}^{(l)} \left(\mathbf{W}^{(1)}\right)^+ \quad (\mathbf{W}^+ \mathbf{W} - \mathbf{I}_p) \right] \in \mathbb{R}^{p \times 2p}$.*

1. When $m = k$, there is no distilled data $\mathbf{X}_S$ that can guarantee $\mathbf{W}_S = \mathbf{W}$ in general useless the columns of $\mathbf{W}^+ \mathbf{Y}_S$ are in the range space of $\prod_{l=1}^{L} \mathbf{W}^{(l)}$.

2. When $m > k$, If $\mathbf{H}$ is full rank and its right singular vectors $\mathbf{V}_\mathbf{H} \in \mathbb{R}^{2p \times 2p}$'s last $p \times p$ submatrix is full rank, then there exists a $\mathbf{X}_S$ such that $\mathbf{W}_S = \mathbf{W}$.

*Proof sketch.* To find the distilled data theoretically, we need to guarantee 1) the feature $\phi(\mathbf{X}_S)$ need to guarantee $\mathbf{W}_S = \mathbf{W}$ and 2) there are some distilled data $\mathbf{X}_S$ corresponding to the feature. For the first condition, Theorem 5.1 gives the sufficient and necessary condition of $\phi(\mathbf{X}_S)$. However, the formulation involves a pseudoinverse of sum of matrices, which does not have a concise formulation and therefore is hard to handle when solving $\mathbf{X}_S$. Instead, in Theorem C.2, we derive a sufficient condition of $\phi(\mathbf{X}_S)$ without pseudoinverse. For the second condition, $\phi(\mathbf{X}_S) = \phi^*$ for a given $\phi^*$ is an overdetermined system of linear equations of $\mathbf{X}_S$. We find the formulation of $\phi(\mathbf{X}_S)$ such that the overdetermined system has solutions. Then combining the two conditions together, we end up with an equation that has multiple free variables. Combing the variables together and solving the equation will give us the results. In the proof, we provide the algorithm to compute the distilled data when the assumptions are satisfied. $\square$

# 6 Applications

In this section, we show that our theoretical results can be useful in some applications. We first show the conditions in Theorem 5.1 can be necessary or sufficient conditions for KIP-type algorithms to converge in Sec 6.1. We also show our distilled dataset for KRR can provably preserve the privacy of the original dataset in Sec 6.2.

## 6.1 An Implication for KIP-type Algorithms

In KIP [25], FRePo [40], RFAD [17], and RCIG [18], a loss function as follows is optimized:

$$L(\mathbf{X}_S) = \|\mathbf{W}_S \phi(\mathbf{X}) - \mathbf{Y}\|_F^2 = \left\| \mathbf{Y}_S \left( K(\mathbf{X}_S, \mathbf{X}_S) + \lambda_S \mathbf{I}_m \right)^{-1} K(\mathbf{X}_S, \mathbf{X}) - \mathbf{Y} \right\|_F^2. \quad (1)$$

Below we show our results can be sufficient conditions for the above loss function to converge to 0 even if the loss function is highly non-convex.

**Theorem 6.1.** *Suppose $\phi(\mathbf{X})$ is full rank and $\mathbf{W}$ is computed with $\lambda = 0$. Then*

1. *when $n \leq p$, the $\phi(\mathbf{X}_S)$ that can guarantee $\mathbf{W}_S = \mathbf{W}$ in Theorem 5.1 is sufficient for $L(\mathbf{X}_S) = 0$.*

2. *when $n \geq p$, the $\phi(\mathbf{X}_S)$ that can guarantee $\mathbf{W}_S = \mathbf{W}$ in Theorem 5.1 is necessary for $L(\mathbf{X}_S) = 0$.*

From this theorem, we see that KIP-type algorithms are enforcing $\mathbf{W}_S = \mathbf{W}$ to some extent. While the KIP algorithm is computationally expensive, our solution for $\phi(\mathbf{X}_S)$ can be computed efficiently and directly utilized in KIP-type algorithms for efficient optimization.

## 6.2 Privacy Preservation of Dataset Distillation

For our dataset distillation algorithm for KRR, we show that the original dataset cannot be recovered from the distilled dataset, therefore provably preserving the privacy of the original dataset while having the performance guarantees at the same time.

**Proposition 6.1.** *Suppose $n > k$ and $\mathbf{Y}$ is rank $k$. Given $\lambda_S, \phi$, for a distilled dataset $(\mathbf{X}_S, \mathbf{Y}_S)$ that can guarantee $\mathbf{W}_S = \mathbf{W}$ in Theorem 5.1, we can reconstruct $\mathbf{W}$ from $\phi(\mathbf{X}_S)$. However, given $\mathbf{W}$, there are infinitely many solutions for $\phi(\mathbf{X})$.*

Since there are infinitely many solutions for $\phi(\mathbf{X})$, it is impossible to recover $\phi(\mathbf{X})$ without additional information. As long as $\phi$ does not contain any information of $\mathbf{X}$, then $\mathbf{X}$ will not be able to recover from distilled dataset $(\mathbf{X}_S, \mathbf{Y}_S)$. Note in Sec. 4.2, we use additional information (real images as reference points) to compute $\mathbf{X}_S$. Therefore $\mathbf{X}_S$ resembles original images and we may recover these original images from $\mathbf{X}_S$. However, if we generate the distilled data with random noise, the distilled data will contain no additional information and protect the privacy of the original dataset. In

summary, we can control whether to generate realistic data by using real data as reference points or protect privacy by generating noisy distilled data as shown in Figure 2.

More formally, we prove that the distilled data can be differential private with respect to the original dataset if we take $\mathbf{Z}$ to be random Gaussian with suitable variance.

**Theorem 6.2.** *Under the same setting of Theorem 4.1, suppose that $\lambda = 0$, all data are bounded $\|\boldsymbol{x}_i\|_2 \leq B$, and the smallest singular value of the original datasets is bounded from below $\sigma_{min}(\mathbf{X}) > \sigma_0$. Suppose $\mathbf{Y}_S$ is independent of $\mathbf{X}$ and unknown to the adversary. Let $[\mathbf{Y}_S^+]_i$ denote its $i$-th row. Let $\epsilon, \delta \in (0,1)$ and take the elements of $\mathbf{Z} \sim \mathcal{N}(0, \sigma^2)$ with $\sigma \geq \max_{i \in [m]} \frac{2\sqrt{\ln(1.25/\delta)}B\|[\mathbf{Y}_S^+]_i\mathbf{Y}\|_2}{\sigma_0^2\epsilon\|[\mathbf{I}_m - \mathbf{Y}_S^+\mathbf{Y}_S]_i\|_2}$, then each row of $\mathbf{X}_S$ is $(\epsilon, \delta)$-differential private with respect to $\mathbf{X}$.*

## 7 Experiments

**(I) Analytical Computation of Dataset Distillation.** In Table 3, we verify our theory of dataset distillation for LRR and KRR with subjective mapping. We compute the distilled dataset for different models using Algorithm 1. The models are KRR with different feature mappings: 1) identity mapping (linear model), 2) one-hidden-layer LeakyReLU neural network, and 3) Random Fourier Features (RFF) of Gaussian kernel. The feature mappings are constructed such that the feature dimension is equal to the data dimension, i.e. $p = d$. For NNs, we use random initialized ones and use the activations as feature mappings. As increasing the depth of NNs does not improve the performance, we only use one hidden layer. For simplicity, we set $\lambda_S = 0$ for all experiments. To choose the original model's regularization $\lambda$, we split the original training set into a training set and a validation set, and choose the $\lambda$ that performs best on the validation set. The results show that our analytically computed distilled dataset can indeed recover the original models' parameters and performance. Some slight differences are caused by the numerical error in recovering the data from features and computing the KRR solutions. As the purpose of this experiment is to verify if the distilled dataset can recover the performance of a specific original model, we did not report the error bars.

Table 3: Verification of our theory. Test accuracy of original models and models trained on the distilled dataset. IPC: images per class.

| Dataset | IPC | Linear | FCNN | RFF |
|---|---|---|---|---|
| MNIST | Original model | 86.41 | 93.89 | 93.82 |
| | 1 | 86.41 | 93.89 | 93.82 |
| | 10 | 86.41 | 93.89 | 93.82 |
| | 50 | 86.41 | 93.85 | 93.82 |
| CIFAR-10 | Original model | 39.48 | 47.86 | 42.84 |
| | 1 | 39.48 | 47.87 | 42.84 |
| | 10 | 39.48 | 47.84 | 42.87 |
| | 50 | 39.48 | 47.81 | 42.73 |
| CIFAR-100 | Original model | 14.37 | 21.42 | 18.71 |
| | 1 | 14.37 | 21.41 | 18.70 |
| | 10 | 14.37 | 21.52 | 18.69 |
| | 50 | 14.37 | 21.49 | 18.57 |

**(II) Comparison with KIP.** In Table 4, we compare our algorithm with KIP in terms of performance and efficiency under the setting of a subjective mapping. The test accuracy of models trained on distilled datasets and averaged computational cost (GPU Seconds) are reported. The mean and standard deviation of test accuracy are computed over four independent runs. As the experiment (I), we use a randomly initialized one-hidden-layer LeakyReLU NN with $p = d$ as the feature mapping. For KIP, we implement their algorithm where we optimize a loss function (1) and use label distillation at each training step. For our results, we compute the distilled dataset using Algorithm 1. Our algorithm performs better than KIP on CIFAR-10 and CIFAR-100 while being significantly more efficient. We did not report the result of KIP with IPC=50 on CIFAR-100 because the estimated running time is more than 110 hours.

This experiment mainly aims to show that our theoretical guarantee can be transferred to practice. As the proposed Algorithm 1 is mainly for KRR with surjective mappings, we verified it and compared it with baselines in this setting. We use a randomly initialized bijective NN in order to match previous algorithms that use a randomly initialized NN. If a pre-trained NN is used, the accuracy can be improved and may match the SOTA.

Table 4: Comparison between our algorithm and KIP.

| Dataset | IPC | KIP [25] | | Ours | | |
|---|---|---|---|---|---|---|
| | | Accuracy ↑ | Cost ↓ (GPU Sec.) | Accuracy ↑ | Cost ↓ (GPU Sec.) | Speedup over KIP ↑ |
| MNIST | 1 | 93.44±0.17 | 159 | **93.72±0.14** | 16 | **9.9×** |
| | 10 | **93.75±0.10** | 554 | 93.69±0.17 | 16 | **34.6×** |
| | 50 | **93.72±0.11** | 3114 | 93.62±0.24 | 16 | **194.6×** |
| CIFAR-10 | 1 | 45.83±0.29 | 225 | **47.85±0.10** | 21 | **10.7×** |
| | 10 | 47.50±0.29 | 594 | **47.76±0.12** | 20 | **29.7×** |
| | 50 | 47.48±0.20 | 3510 | **47.77±0.06** | 20 | **175.5×** |
| CIFAR-100 | 1 | 20.08±0.20 | 616 | **21.58±0.15** | 20 | **30.8×** |
| | 10 | 21.56±0.16 | 9323 | **21.59±0.15** | 20 | **466.1×** |
| | 50 | - | ∼396000 | **21.58±0.13** | 25 | **∼15840.0×** |

**(III) Privacy Protection.** In this experiment, we show our algorithm can be used to protect the privacy of the original dataset. Same as experiment (II), we use a one-hidden-layer LeakyReLU neural network with $p = d$ as the feature mapping and train a KRR model on the original dataset. Then we distill the dataset using Algorithm 1 and generate the distilled data with random Gaussian noise. As shown in Figure 3, the distilled data for MNIST are essentially random noise, which protects the privacy of the original MNIST dataset. At the same time, the model trained on it can recover the original model's performance of 93.87% test accuracy.

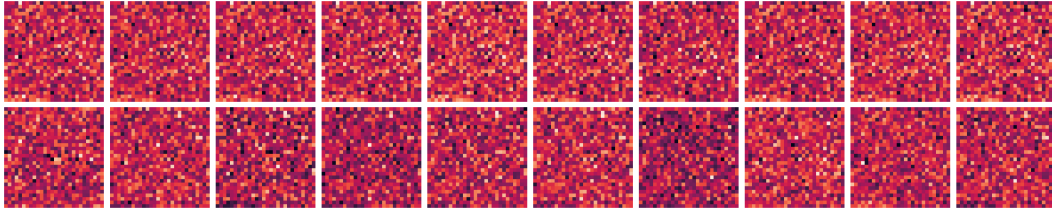

Figure 3: Distilled images of MNIST generated from random noise for a two-layer neural network. $m = 10$ (first row) and 100 (second row).

## 8   Conclusion and Future Works

In this paper, by focusing on dataset distillation for KRR, we show that one data point per class is already necessary and sufficient to recover the original model's performance in many settings. For linear ridge regression and KRR with surjective feature mappings, we provide necessary and sufficient conditions for the distilled dataset to recover the original model's parameters. For KRR with injective feature mappings of deep neural networks, we show that while one data point per class is not sufficient in general, $k + 1$ data points can be sufficient for deep linear neural networks. Our theoretical results facilitate the direct construction of analytical solutions for distilled datasets, leading to a provable and efficient dataset distillation algorithm for KRR. Additionally, we have developed applications for KIP-type algorithms and privacy protection.

Several future research directions are worth exploring. First, while the current analysis shows that $k$ data points are generally insufficient for non-surjective deep non-linear neural networks, determining the minimum number of distilled data points required remains an open question worthy of investigation. Second, this paper focuses on KRR with fixed feature mappings, which differs from some empirical works that train all neural network parameters. Extending the analysis to learnable feature mappings would bridge this gap and provide further insights.

## Acknowledgement

The authors thank the anonymous reviewers for valuable feedback on the manuscript. T.-W. Weng is supported by National Science Foundation awards CCF-2107189, IIS-2313105, IIS-2430539, the Hellman Fellowship, and Intel Rising Star Faculty Award.

## References

[1] Nicholas Carlini, Vitaly Feldman, and Milad Nasr. No free lunch in" privacy for free: How does dataset condensation help privacy". *arXiv preprint arXiv:2209.14987*, 2022.

[2] George Cazenavette, Tongzhou Wang, Antonio Torralba, Alexei A Efros, and Jun-Yan Zhu. Dataset distillation by matching training trajectories. In *Proceedings of the IEEE/CVF Conference on Computer Vision and Pattern Recognition*, pages 4750–4759, 2022.

[3] Randall E Cline. Representations for the generalized inverse of sums of matrices. *Journal of the Society for Industrial and Applied Mathematics, Series B: Numerical Analysis*, 2(1):99–114, 1965.

[4] Justin Cui, Ruochen Wang, Si Si, and Cho-Jui Hsieh. Scaling up dataset distillation to imagenet-1k with constant memory. In *International Conference on Machine Learning*, pages 6565–6590. PMLR, 2023.

[5] Tian Dong, Bo Zhao, and Lingjuan Lyu. Privacy for free: How does dataset condensation help privacy? In *International Conference on Machine Learning*, pages 5378–5396. PMLR, 2022.

[6] Cynthia Dwork, Aaron Roth, et al. The algorithmic foundations of differential privacy. *Foundations and Trends® in Theoretical Computer Science*, 9(3–4):211–407, 2014.

[7] Yunzhen Feng, Shanmukha Ramakrishna Vedantam, and Julia Kempe. Embarrassingly simple dataset distillation. In *The Twelfth International Conference on Learning Representations*, 2024. URL https://openreview.net/forum?id=PLoWVP7Mjc.

[8] Ziyao Guo, Kai Wang, George Cazenavette, HUI LI, Kaipeng Zhang, and Yang You. Towards lossless dataset distillation via difficulty-aligned trajectory matching. In *The Twelfth International Conference on Learning Representations*, 2024. URL https://openreview.net/forum?id=rTBL8OhdhH.

[9] Zachary Izzo and James Zou. A theoretical study of dataset distillation. In *NeurIPS 2023 Workshop on Mathematics of Modern Machine Learning*, 2023. URL https://openreview.net/forum?id=dq5QGXGxoJ.

[10] Arthur Jacot, Franck Gabriel, and Clément Hongler. Neural tangent kernel: Convergence and generalization in neural networks. *Advances in neural information processing systems*, 31, 2018.

[11] Jang-Hyun Kim, Jinuk Kim, Seong Joon Oh, Sangdoo Yun, Hwanjun Song, Joonhyun Jeong, Jung-Woo Ha, and Hyun Oh Song. Dataset condensation via efficient synthetic-data parameterization. In *International Conference on Machine Learning*, pages 11102–11118. PMLR, 2022.

[12] Alex Krizhevsky, Geoffrey Hinton, et al. Learning multiple layers of features from tiny images. 2009.

[13] Yann LeCun, Léon Bottou, Yoshua Bengio, and Patrick Haffner. Gradient-based learning applied to document recognition. *Proceedings of the IEEE*, 86(11):2278–2324, 1998.

[14] Jaehoon Lee, Yasaman Bahri, Roman Novak, Samuel S Schoenholz, Jeffrey Pennington, and Jascha Sohl-Dickstein. Deep neural networks as gaussian processes. *arXiv preprint arXiv:1711.00165*, 2017.

[15] Saehyung Lee, Sanghyuk Chun, Sangwon Jung, Sangdoo Yun, and Sungroh Yoon. Dataset condensation with contrastive signals. In *International Conference on Machine Learning*, pages 12352–12364. PMLR, 2022.

[16] Zhu Li, Jean-Francois Ton, Dino Oglic, and Dino Sejdinovic. Towards a unified analysis of random fourier features. *Journal of Machine Learning Research*, 22(108):1–51, 2021.

[17] Noel Loo, Ramin Hasani, Alexander Amini, and Daniela Rus. Efficient dataset distillation using random feature approximation. *Advances in Neural Information Processing Systems*, 35: 13877–13891, 2022.

[18] Noel Loo, Ramin Hasani, Mathias Lechner, and Daniela Rus. Dataset distillation with convexified implicit gradients. In *International Conference on Machine Learning*, pages 22649–22674. PMLR, 2023.

[19] Alaa Maalouf, Ibrahim Jubran, and Dan Feldman. Fast and accurate least-mean-squares solvers. *Advances in Neural Information Processing Systems*, 32, 2019.

[20] Alaa Maalouf, Ibrahim Jubran, and Dan Feldman. Fast and accurate least-mean-squares solvers for high dimensional data. *IEEE Transactions on Pattern Analysis and Machine Intelligence*, 44(12):9977–9994, 2022.

[21] Alaa Maalouf, Murad Tukan, Noel Loo, Ramin Hasani, Mathias Lechner, and Daniela Rus. On the size and approximation error of distilled sets. *arXiv preprint arXiv:2305.14113*, 2023.

[22] G. Marsaglia and G. P. H. Styan. When does rank(a+b)=rank(a)+rank(b)? *Canadian Mathematical Bulletin*, 15(3):451–452, 1972. doi: 10.4153/CMB-1972-082-8.

[23] Lingsheng Meng and Bing Zheng. The optimal perturbation bounds of the moore–penrose inverse under the frobenius norm. *Linear algebra and its applications*, 432(4):956–963, 2010.

[24] Timothy Nguyen, Zhourong Chen, and Jaehoon Lee. Dataset meta-learning from kernel ridge-regression. *arXiv preprint arXiv:2011.00050*, 2020.

[25] Timothy Nguyen, Roman Novak, Lechao Xiao, and Jaehoon Lee. Dataset distillation with infinitely wide convolutional networks. *Advances in Neural Information Processing Systems*, 34:5186–5198, 2021.

[26] Adam Paszke, Sam Gross, Francisco Massa, Adam Lerer, James Bradbury, Gregory Chanan, Trevor Killeen, Zeming Lin, Natalia Gimelshein, Luca Antiga, et al. Pytorch: An imperative style, high-performance deep learning library. *Advances in neural information processing systems*, 32, 2019.

[27] Ali Rahimi and Benjamin Recht. Random features for large-scale kernel machines. *Advances in neural information processing systems*, 20, 2007.

[28] Olga Russakovsky, Jia Deng, Hao Su, Jonathan Krause, Sanjeev Satheesh, Sean Ma, Zhiheng Huang, Andrej Karpathy, Aditya Khosla, Michael Bernstein, Alexander C. Berg, and Li Fei-Fei. ImageNet Large Scale Visual Recognition Challenge. *International Journal of Computer Vision (IJCV)*, 115(3):211–252, 2015. doi: 10.1007/s11263-015-0816-y.

[29] Noveen Sachdeva and Julian McAuley. Data distillation: A survey. *Transactions on Machine Learning Research*, 2023. ISSN 2835-8856. URL https://openreview.net/forum?id=lmXMXP74TO. Survey Certification.

[30] James R Schott. *Matrix analysis for statistics*. John Wiley & Sons, 2016.

[31] Felipe Petroski Such, Aditya Rawal, Joel Lehman, Kenneth Stanley, and Jeffrey Clune. Generative teaching networks: Accelerating neural architecture search by learning to generate synthetic training data. In *International Conference on Machine Learning*, pages 9206–9216. PMLR, 2020.

[32] Murad Tukan, Alaa Maalouf, and Margarita Osadchy. Dataset distillation meets provable subset selection. *arXiv preprint arXiv:2307.08086*, 2023.

[33] Kai Wang, Bo Zhao, Xiangyu Peng, Zheng Zhu, Shuo Yang, Shuo Wang, Guan Huang, Hakan Bilen, Xinchao Wang, and Yang You. Cafe: Learning to condense dataset by aligning features. In *Proceedings of the IEEE/CVF Conference on Computer Vision and Pattern Recognition*, pages 12196–12205, 2022.

[34] Tongzhou Wang, Jun-Yan Zhu, Antonio Torralba, and Alexei A Efros. Dataset distillation. *arXiv preprint arXiv:1811.10959*, 2018.

[35] Enneng Yang, Li Shen, Zhenyi Wang, Tongliang Liu, and Guibing Guo. An efficient dataset condensation plugin and its application to continual learning. *Advances in Neural Information Processing Systems*, 36, 2023.

[36] Zeyuan Yin, Eric Xing, and Zhiqiang Shen. Squeeze, recover and relabel: Dataset condensation at imagenet scale from a new perspective. *Advances in Neural Information Processing Systems*, 36, 2024.

[37] Bo Zhao and Hakan Bilen. Dataset condensation with differentiable siamese augmentation. In *International Conference on Machine Learning*, pages 12674–12685. PMLR, 2021.

[38] Bo Zhao and Hakan Bilen. Dataset condensation with distribution matching. In *Proceedings of the IEEE/CVF Winter Conference on Applications of Computer Vision*, pages 6514–6523, 2023.

[39] Bo Zhao, Konda Reddy Mopuri, and Hakan Bilen. Dataset condensation with gradient matching. *arXiv preprint arXiv:2006.05929*, 2020.

[40] Yongchao Zhou, Ehsan Nezhadarya, and Jimmy Ba. Dataset distillation using neural feature regression. *Advances in Neural Information Processing Systems*, 35:9813–9827, 2022.

# Appendices

---

**Algorithm 1** Dataset distillation for kernel ridge regression

---

**Input:** Number of distilled data $m$, number of classes $k$, regularization $\lambda_S$, feature mapping $\phi$, original model's parameter $\mathbf{W}$, generate from original data or random noise

 1: **if** Generate from original data **then**

 2:     Sample $m$ balanced initialized data $\hat{\mathbf{X}}_S$ from original dataset. Initialize $\mathbf{Y}_S$ as corresponding one-hot labels

 3:     Compute $\phi_{\lambda_S}(\hat{\mathbf{X}}_S)$

 4:     **if** $\mathrm{Rank}\left(\mathbf{W}\phi_{\lambda_S}(\hat{\mathbf{X}}_S)\right) = k$ **then**

 5:         $\mathbf{Y}_S = \mathbf{W}\phi_{\lambda_S}(\hat{\mathbf{X}}_S)$

 6:     **end if**

 7:     Construct $\mathbf{Z} = \left(\mathbf{I}_m - \mathbf{Y}_S^+\mathbf{Y}_S\right)\left(\phi_{\lambda_S}(\hat{\mathbf{X}}_S)^+ - \mathbf{Y}_S^+\mathbf{W}\right)$.

 8: **else if** Generate from random noise **then**

 9:     Sample $\mathbf{Z}$ from random noise. Initialize $\mathbf{Y}_S$ as balanced one-hot labels

10: **end if**

11: Compute $\mathbf{D} = \mathbf{Y}_S^+\mathbf{W} + \left(\mathbf{I}_m - \mathbf{Y}_S^+\mathbf{Y}_S\right)\mathbf{Z}$

12: **if** $\lambda_S > 0$ **then**

13:     Compute the SVD of $\mathbf{D}$, $\mathbf{D} = \mathbf{V}\mathrm{diag}(\sigma_1', \ldots, \sigma_r')\mathbf{U}^\top$

14:     **if** $\lambda_S > \frac{1}{4\sigma_1'^2}$ **then**

15:         $\lambda_S = \frac{1}{4\sigma'^2}$

16:     **end if**

17:     Construct $\phi(\mathbf{X}_S) = \mathbf{U}\mathrm{diag}(\sigma_1, \ldots, \sigma_r)\mathbf{V}^\top$ where $\sigma_i = 0$ if $\sigma_i' = 0$ and $\sigma_i = \frac{1 \pm \sqrt{1 - 4\lambda_S\sigma_i'^2}}{2\sigma_i'}$ otherwise.

18: **else if** $\lambda_S = 0$ **then**

19:     Construct $\phi(\mathbf{X}_S) = \mathbf{D}^+$

20: **end if**

21: Construct $\mathbf{X}_S$ from $\phi(\mathbf{X}_S)$

**Output:** $\mathbf{X}_S, \lambda_S$

---

## A   Broader Impact

Our approach can be used to protect data privacy, which may have a positive societal impact. There are no particular ethical concerns we are aware of.

## B   Additional Experiment Details

All the experiments are implemented with PyTorch [26] and conducted on a single 24G A5000 GPU.

# C   Proofs for Linear Ridge Regularization

## C.1   Analytical Computation for Linear Ridge Regression

**Theorem 4.1.** *When $m < k$, there is no $\mathbf{X}_S$ can guarantee $\mathbf{W}_S = \mathbf{W}$ unless the columns of $\mathbf{W}$ are in the range space of $\mathbf{Y}_S$. When $m \geq k$ and $\mathbf{Y}_S$ is rank $k$, let $r = \min(m, d)$ and take $\mathbf{D} = \mathbf{Y}_S^+ \mathbf{W} + \left( \mathbf{I}_m - \mathbf{Y}_S^+ \mathbf{Y}_S \right) \mathbf{Z}$, where $\mathbf{Z} \in \mathbb{R}^{m \times d}$ is any matrix of the same size as $\mathbf{X}_S^\top$. Suppose the reduced SVD of $\mathbf{D}$ is $\mathbf{D} = \mathbf{V} diag(\sigma_1', \ldots, \sigma_r') \mathbf{U}^\top$ with $\sigma_1' \geq \cdots \geq \sigma_r' \geq 0$, the following results hold:*

*1. $\lambda_S > 0$: $\mathbf{W}_S = \mathbf{W}$ if and only if, for any $\mathbf{D}$ defined above, $\lambda_S \leq \frac{1}{4\sigma_1'^2}$ and $\mathbf{X}_S = \mathbf{U} diag(\sigma_1, \ldots, \sigma_r) \mathbf{V}^\top$ where $\sigma_i = \begin{cases} 0, & \text{if } \sigma_i' = 0, \\ \frac{1 \pm \sqrt{1 - 4\lambda_S \sigma_i'^2}}{2\sigma_i'}, & \text{otherwise.} \end{cases}$*

*2. $\lambda_S = 0$: $\mathbf{W}_S = \mathbf{W}$ if and only if $\mathbf{X}_S = \mathbf{D}^+$ for any $\mathbf{D}$ defined above.*

*Proof.* Recall $\mathbf{W}_S = \mathbf{Y}_S \mathbf{X}_{\lambda_S}^+$ where

$$\mathbf{X}_{\lambda_S}^+ = \begin{cases} \left( \mathbf{X}_S^\top \mathbf{X}_S + \lambda_S \mathbf{I}_m \right)^{-1} \mathbf{X}_S^\top = \mathbf{X}_S^\top \left( \mathbf{X}_S \mathbf{X}_S^\top + \lambda_S \mathbf{I}_d \right)^{-1}, & \text{if } \lambda_S > 0, \\ \mathbf{X}_S^+, & \text{if } \lambda_S = 0. \end{cases}$$

Let $\mathbf{W}_S = \mathbf{W}$,

$$\mathbf{Y}_S \mathbf{X}_{\lambda_S}^+ = \mathbf{W}.$$

When $m < k$, this is an overdetermined system for $\mathbf{X}_{\lambda_S}^+$. There is no solution for $\mathbf{X}_{\lambda_S}^+$ in general therefore no solution for $\mathbf{X}_S$ unless all the columns of $\mathbf{W}$ are in the range space of $\mathbf{Y}_S$. When there is a solution, we can solve it as following $m \geq k$ cases.

In the following, we consider the $m \geq k$ case. Since $m \geq k$ and $\mathbf{Y}_S$ is rank $k$, the solutions of $\mathbf{X}_{\lambda_S}^+$ are given by

$$\mathbf{X}_{\lambda_S}^+ = \mathbf{Y}_S^+ \mathbf{W} + \left( \mathbf{I}_m - \mathbf{Y}_S^+ \mathbf{Y}_S \right) \mathbf{Z} \tag{2}$$

where $\mathbf{Z} \in \mathbb{R}^{m \times d}$ is any matrix of the same size as $\mathbf{X}_{\lambda_S}^+$. When $k = m$, the solution is unique $\mathbf{X}_{\lambda_S}^+ = \mathbf{Y}_S^{-1} \mathbf{W}$. However, there are solutions for $\mathbf{X}_{\lambda_S}^+$ does not mean there are solutions for $\mathbf{X}_S$. Next, we need to solve $\mathbf{X}_S$ from $\mathbf{X}_{\lambda_S}^+$.

**1.** When $m \leq d$. Suppose the reduced SVD of $\mathbf{X}_S$ is $\mathbf{X}_S = \mathbf{U} \mathbf{\Sigma} \mathbf{V}^\top$, where $\mathbf{\Sigma} = diag(\sigma_1, \ldots, \sigma_m) \in \mathbb{R}^{m \times m}$, $\mathbf{V} \in \mathbb{R}^{m \times m}$ is a unitary matrix and $\mathbf{U} \in \mathbb{R}^{d \times m}$ is the first $m$ columns of a unitary matrix. Then when $\lambda_S > 0$,

$$\begin{aligned} \mathbf{X}_{\lambda_S}^+ &= \left( \mathbf{X}_S^\top \mathbf{X}_S + \lambda_S \mathbf{I}_m \right)^{-1} \mathbf{X}_S^\top \\ &= \left( \mathbf{V} \mathbf{\Sigma} \mathbf{U}^\top \mathbf{U} \mathbf{\Sigma} \mathbf{V}^\top + \lambda_S \mathbf{I}_m \right)^{-1} \mathbf{V} \mathbf{\Sigma} \mathbf{U}^\top \\ &= \left( \mathbf{V} \mathbf{\Sigma}^2 \mathbf{V}^\top + \lambda_S \mathbf{I}_m \right)^{-1} \mathbf{V} \mathbf{\Sigma} \mathbf{U}^\top \\ &= \left( \mathbf{V} \left( \mathbf{\Sigma}^2 + \lambda_S \mathbf{I}_m \right) \mathbf{V}^\top \right)^{-1} \mathbf{V} \mathbf{\Sigma} \mathbf{U}^\top \\ &= \mathbf{V} \left( \mathbf{\Sigma}^2 + \lambda_S \mathbf{I}_m \right)^{-1} \mathbf{V}^\top \mathbf{V} \mathbf{\Sigma} \mathbf{U}^\top \\ &= \mathbf{V} \left( \mathbf{\Sigma}^2 + \lambda_S \mathbf{I}_m \right)^{-1} \mathbf{\Sigma} \mathbf{U}^\top \\ &= \mathbf{V} diag \left( \frac{\sigma_1}{\sigma_1^2 + \lambda_S}, \ldots, \frac{\sigma_m}{\sigma_m^2 + \lambda_S} \right) \mathbf{U}^\top. \end{aligned} \tag{3}$$

Combining (2) and (3), we must have

$$\mathbf{V} diag \left( \frac{\sigma_1}{\sigma_1^2 + \lambda_S}, \ldots, \frac{\sigma_m}{\sigma_m^2 + \lambda_S} \right) \mathbf{U}^\top = \mathbf{Y}_S^+ \mathbf{W} + \left( \mathbf{I}_m - \mathbf{Y}_S^+ \mathbf{Y}_S \right) \mathbf{Z}.$$

Denote $\mathbf{D} = \mathbf{Y}_S^+ \mathbf{W} + \left( \mathbf{I}_m - \mathbf{Y}_S^+ \mathbf{Y}_S \right) \mathbf{Z}$. Given $\mathbf{D}$, we can compute its reduced SVD $\mathbf{D} = \mathbf{V}' diag(\sigma_1', \ldots, \sigma_m') \mathbf{U}'^\top$ with $\sigma_1' \geq \cdots \geq \sigma_m'$. Note that SVD of a matrix is unique. Since

$\mathbf{D} = \mathbf{V}'\mathrm{diag}(\sigma'_1,\ldots,\sigma'_m)\mathbf{U}'^\top = \mathbf{V}\mathrm{diag}(\frac{\sigma_1}{\sigma_1^2+\lambda_S},\ldots,\frac{\sigma_m}{\sigma_m^2+\lambda_S})\mathbf{U}^\top$, we must have $\mathbf{V} = \mathbf{V}'$, $\mathbf{U} = \mathbf{U}'$, and

$$\sigma'_i = \frac{\sigma_i}{\sigma_i^2 + \lambda_S}$$

That is

$$\sigma'_i\sigma_i^2 - \sigma_i + \lambda_S\sigma'_i = 0$$

When $\sigma'_i = 0$, we have $\sigma_i = 0$. When $\sigma'_i \neq 0$ and $\lambda_S \leq \frac{1}{4\sigma_1'^2} \leq \frac{1}{4\sigma_i'^2}$, it has solutions given by

$$\sigma_i = \frac{1 \pm \sqrt{1 - 4\lambda_S\sigma_i'^2}}{2\sigma'_i}$$

Take the above computed $\mathbf{U}, \mathbf{V}$, and $\mathbf{\Sigma}$, we can construct $\mathbf{X}_S$. Above shows such $\mathbf{X}_S$ is a necessary condition for $\mathbf{W}_S = \mathbf{W}$. To show the sufficiency, take such $\mathbf{X}_S$ into $\mathbf{W}_S$.

$$\begin{aligned}
\mathbf{W}_S &= \mathbf{Y}_S\mathbf{V}\mathrm{diag}(\frac{\sigma_1}{\sigma_1^2 + \lambda_S},\ldots,\frac{\sigma_m}{\sigma_m^2 + \lambda_S})\mathbf{U}^\top \\
&= \mathbf{Y}_S\mathbf{V}\mathrm{diag}(\sigma'_1,\ldots,\sigma'_m)\mathbf{U}^\top \\
&= \mathbf{Y}_S\mathbf{D} \\
&= \mathbf{Y}_S\left(\mathbf{Y}_S^+\mathbf{W} + \left(\mathbf{I}_m - \mathbf{Y}_S^+\mathbf{Y}_S\right)\mathbf{Z}\right) \\
&= \mathbf{W}
\end{aligned}$$

which shows it is a sufficient condition.

When $\lambda_S = 0$, we have $\mathbf{W}_S = \mathbf{Y}_S\mathbf{X}_S^+$. Let $\mathbf{W}_S = \mathbf{Y}_S\mathbf{X}_S^+ = \mathbf{W}$. The solution for $\mathbf{X}_S^+$ is

$$\mathbf{X}_S^+ = \mathbf{Y}_S^+\mathbf{W} + \left(\mathbf{I}_m - \mathbf{Y}_S^+\mathbf{Y}_S\right)\mathbf{Z}$$

where $\mathbf{Z} \in \mathbb{R}^{m \times d}$ is any matrix of the same size as $\mathbf{X}_S^\top$. Therefore

$$\mathbf{X}_S = \left(\mathbf{Y}_S^+\mathbf{W} + \left(\mathbf{I}_m - \mathbf{Y}_S^+\mathbf{Y}_S\right)\mathbf{Z}\right)^+.$$

Similarly, this is a necessary condition for $\mathbf{W}_S = \mathbf{W}$. To show the sufficiency, take such $\mathbf{X}_S$ into $\mathbf{W}_S$.

$$\begin{aligned}
\mathbf{W}_S &= \mathbf{Y}_S\left(\mathbf{Y}_S^+\mathbf{W} + \left(\mathbf{I}_m - \mathbf{Y}_S^+\mathbf{Y}_S\right)\mathbf{Z}\right) \\
&= \mathbf{W}
\end{aligned}$$

which shows it is a sufficient condition.

**2.** When $m > d$. Suppose the reduced SVD of $\mathbf{X}_S$ is $\mathbf{X}_S = \mathbf{U}\mathbf{\Sigma}\mathbf{V}^\top$, where $\mathbf{\Sigma} = \mathrm{diag}(\sigma_1,\ldots,\sigma_d) \in \mathbb{R}^{d \times d}$, $\mathbf{U} \in \mathbb{R}^{d \times d}$ is a unitary matrix and $\mathbf{V} \in \mathbb{R}^{m \times d}$ is the first $d$ columns of a unitary matrix. Then when $\lambda_S > 0$,

$$\begin{aligned}
\mathbf{X}_{\lambda_S}^+ &= \mathbf{X}_S^\top\left(\mathbf{X}_S\mathbf{X}_S^\top + \lambda_S\mathbf{I}_d\right)^{-1} \\
&= \mathbf{V}\mathbf{\Sigma}\mathbf{U}^\top\left(\mathbf{U}\mathbf{\Sigma}\mathbf{V}^\top\mathbf{V}\mathbf{\Sigma}\mathbf{U}^\top + \lambda_S\mathbf{I}_d\right)^{-1} \\
&= \mathbf{V}\mathbf{\Sigma}\mathbf{U}^\top\left(\mathbf{U}\mathbf{\Sigma}^2\mathbf{U}^\top + \lambda_S\mathbf{I}_d\right)^{-1} \\
&= \mathbf{V}\mathbf{\Sigma}\mathbf{U}^\top\left(\mathbf{U}\left(\mathbf{\Sigma}^2 + \lambda_S\mathbf{I}_d\right)\mathbf{U}^\top\right)^{-1} \\
&= \mathbf{V}\mathbf{\Sigma}\mathbf{U}^\top\mathbf{U}\left(\mathbf{\Sigma}^2 + \lambda_S\mathbf{I}_d\right)^{-1}\mathbf{U}^\top \\
&= \mathbf{V}\mathbf{\Sigma}\left(\mathbf{\Sigma}^2 + \lambda_S\mathbf{I}_d\right)^{-1}\mathbf{U}^\top \\
&= \mathbf{V}\mathrm{diag}(\frac{\sigma_1}{\sigma_1^2 + \lambda_S},\ldots,\frac{\sigma_d}{\sigma_d^2 + \lambda_S})\mathbf{U}^\top \qquad (4)
\end{aligned}$$

Then we proceed similarly to the $m \leq d$ case. Last, we can unify two cases by taking $r = \min(m, d)$.

$\square$

**Proposition 4.1.** *When $m \geq k$ and $\mathbf{Y}_S, \mathbf{W}$ are rank $k$, the $\mathbf{X}_S$ in Theorem 4.1 is full rank for any full-rank $\mathbf{Z}$ such that* $\mathrm{Range}\left(\mathbf{W}^\top\right) \cap \mathrm{Range}\left(\mathbf{Z}^\top\right) = \{\mathbf{0}\}$.

*Proof.* $\mathbf{X}_S$ is computed from $\mathbf{D} = \mathbf{Y}_S^+\mathbf{W} + \left(\mathbf{I}_m - \mathbf{Y}_S^+\mathbf{Y}_S\right)\mathbf{Z}$. It is easy to check that $\mathbf{X}_S$ is full rank if and only if $\mathbf{D}$ is full rank.

When $\mathbf{Y}_S$ and $\mathbf{W}$ are rank $k$, $\text{Rank}(\mathbf{Y}_S^+\mathbf{W}) = k$. Since $\text{Rank}(\mathbf{I}_m - \mathbf{Y}_S^+\mathbf{Y}_S) = m - k$ and $\mathbf{Z}$ is full rank, by Sylvester's rank inequality,

$$\text{Rank}(\left(\mathbf{I}_m - \mathbf{Y}_S^+\mathbf{Y}_S\right)\mathbf{Z}) \geq \text{Rank}(\mathbf{I}_m - \mathbf{Y}_S^+\mathbf{Y}_S) + \text{Rank}(\mathbf{Z}) - m$$
$$= m - k + \min(m, d) - m$$
$$= \min(m, d) - k$$

For $\mathbf{D} = \mathbf{Y}_S^+\mathbf{W} + \left(\mathbf{I}_m - \mathbf{Y}_S^+\mathbf{Y}_S\right)\mathbf{Z}$, since the columns of $\mathbf{Y}_S^+\mathbf{W} \in \text{Range}\left(\mathbf{Y}_S^\top\right)$ and the columns of $\left(\mathbf{I}_m - \mathbf{Y}_S^+\mathbf{Y}_S\right)\mathbf{Z} \in \text{Null}\left(\mathbf{Y}_S\right)$. By the fundamental theorem of linear algebra, $\text{Range}\left(\mathbf{Y}_S^\top\right)$ and $\text{Null}\left(\mathbf{Y}_S\right)$ are orthogonal subspaces of $\mathbb{R}^m$. Therefore $\text{Range}\left(\mathbf{Y}_S^+\mathbf{W}\right) \cap \text{Range}\left(\left(\mathbf{I}_m - \mathbf{Y}_S^+\mathbf{Y}_S\right)\mathbf{Z}\right) = \{\mathbf{0}\}$. This can also be seen from $\left(\mathbf{Y}_S^+\mathbf{W}\right)^\top \left(\mathbf{I}_m - \mathbf{Y}_S^+\mathbf{Y}_S\right)\mathbf{Z} = 0$, which shows their columns are orthogonal to each other. If we have $\text{Range}\left(\mathbf{W}^\top\right) \cap \text{Range}\left(\mathbf{Z}^\top\right) = \{\mathbf{0}\}$, then $\text{Range}\left(\left(\mathbf{Y}_S^+\mathbf{W}\right)^\top\right) \cap \text{Range}\left(\mathbf{Z}^\top\left(\mathbf{I}_m - \mathbf{Y}_S^+\mathbf{Y}_S\right)\right) = \{\mathbf{0}\}$. By [22],

$$\text{Rank}(\mathbf{D}) = \text{Rank}(\mathbf{Y}_S^+\mathbf{W}) + \text{Rank}(\left(\mathbf{I}_m - \mathbf{Y}_S^+\mathbf{Y}_S\right)\mathbf{Z}) \geq k + \min(m, d) - k = \min(m, d).$$

Therefore $\mathbf{D}$ is full rank and $\mathbf{X}_S$ is full rank. $\qquad\square$

### C.2 Characterization of Distilled Data without Pseudoinverse

In the last section, we give analytical solutions for $\mathbf{X}_S$ that can guarantee $\mathbf{W}_S = \mathbf{W}$. However, the expression of $\mathbf{X}_S$ involves some pseudoinverse calculation and the explicit expression of $\mathbf{X}_S$ remains unclear because there is no concise formulation for the pseudoinverse of sum of matrices. In this section, we give some direct characterization for $\mathbf{X}_S$.

Again, supposed its reduced SVD is $\mathbf{X}_S = \mathbf{U}\text{diag}(\sigma_1, \ldots, \sigma_r)\mathbf{V}^\top$, where $r = \min(m, d)$. When $\lambda_S > 0$, from Eq. (3) and Eq. (4), $\mathbf{X}_{\lambda_S}^+ = \mathbf{D} = \mathbf{V}\text{diag}(\frac{\sigma_1}{\sigma_1^2 + \lambda_S}, \ldots, \frac{\sigma_r}{\sigma_r^2 + \lambda_S})\mathbf{U}^\top$ and $\mathbf{X}_{\lambda_S} = \mathbf{U}\text{diag}(\frac{\sigma_1}{\sigma_1^2 + \lambda_S}, \ldots, \frac{\sigma_r}{\sigma_r^2 + \lambda_S})^+\mathbf{V}^\top$, where $\left[\text{diag}(\frac{\sigma_1}{\sigma_1^2 + \lambda_S}, \ldots, \frac{\sigma_r}{\sigma_r^2 + \lambda_S})^+\right]_{i,i} = \sigma_i + \frac{\lambda_S}{\sigma_i}$ if $\sigma_i > 0$ else 0. When $\lambda_S = 0$, $\mathbf{X}_{\lambda_S}^+ = \mathbf{X}_S^+$ and $\mathbf{X}_{\lambda_S} = \mathbf{X}_S$. Given $\mathbf{X}_{\lambda_S}$ and $\lambda_S$, we can easily compute $\mathbf{X}_S$ by SVD. Below we give conditions for $\mathbf{W}_S = \mathbf{W}$ through $\mathbf{X}_{\lambda_S}$.

Suppose the eigenvalues of $\mathbf{X}_{\lambda_S}$ are $\sigma_i'$. Then by the definition of $\mathbf{X}_{\lambda_S}$, $\sigma_i + \frac{\lambda_S}{\sigma_i} = \sigma_i'$ if $\sigma_i > 0$. That is

$$\sigma_i^2 - \sigma_i'\sigma_i + \lambda_S = 0$$

Only when $\sigma_i'^2 \geq 4\lambda_S$, there are solution(s) $\sigma_i = \frac{\sigma_i' \pm \sqrt{\sigma_i'^2 - 4\lambda_S}}{2}$. Therefore, to make sure there is a $\mathbf{X}_S$ corresponds to $\mathbf{X}_{\lambda_S}$, the nonzero singular values of $\mathbf{X}_{\lambda_S}$ need to be larger than or equal to $2\sqrt{\lambda_S}$. When $\lambda_S = 0$, there is no requirement.

**Theorem C.1.** *Suppose $k \leq d$ and $\mathbf{W}$ is rank $k$. Take*

$$\mathbf{X}_{\lambda_S} = \mathbf{W}^+\mathbf{Y}_S + \left(\mathbf{I}_d - \mathbf{W}^+\mathbf{W}\right)\mathbf{Z}',$$

*where $\mathbf{Z}' \in \mathbb{R}^{d \times m}$ is any matrix of the same size as $\mathbf{X}_{\lambda_S}$ such that $\mathbf{X}_{\lambda_S}$ is full rank.*

*1. When $m \leq d$, it is a necessary condition for $\mathbf{W}_S = \mathbf{W}$.*

*2. When $m \geq d$, it is a sufficient condition for $\mathbf{W}_S = \mathbf{W}$.*

*Proof.* **Case 1.** When $m \leq d$, recall that $\mathbf{W}_S = \mathbf{Y}_S\mathbf{X}_{\lambda_S}^+$. Set the parameter to be the same $\mathbf{W}_S = \mathbf{W}$ and try to solve $\mathbf{X}_{\lambda_S}$

$$\mathbf{W}_S = \mathbf{Y}_S\mathbf{X}_{\lambda_S}^+ = \mathbf{W}$$

Multiply $\mathbf{X}_{\lambda_S}$ on both sides. Since $\mathbf{X}_{\lambda_S}$ is full rank and $\mathbf{X}_{\lambda_S}^+\mathbf{X}_{\lambda_S} = \mathbf{I}_m$, we have

$$\mathbf{W}\mathbf{X}_{\lambda_S} = \mathbf{Y}_S$$

Since $k \leq d$ and $\mathbf{W}$ is rank $k$, $\mathbf{X}_S$ has infinite many solutions. The general solutions are

$$\mathbf{X}_{\lambda_S} = \mathbf{W}^+ \mathbf{Y}_S + \left(\mathbf{I}_d - \mathbf{W}^+ \mathbf{W}\right) \mathbf{Z}' \tag{5}$$

where $\mathbf{Z}' \in \mathbb{R}^{d \times m}$ is any matrix of the same size as $\mathbf{X}_S$. Therefore, this is a necessary condition for $\mathbf{W}_S = \mathbf{W}$.

**Case 2.** When $m \geq d$, to show the sufficiency, for any $\mathbf{X}_{\lambda_S} = \mathbf{W}^+ \mathbf{Y}_S + \left(\mathbf{I}_d - \mathbf{W}^+ \mathbf{W}\right) \mathbf{Z}'$,

$$\mathbf{W} \mathbf{X}_{\lambda_S} = \mathbf{W} \left[\mathbf{W}^+ \mathbf{Y}_S + \left(\mathbf{I}_d - \mathbf{W}^+ \mathbf{W}\right) \mathbf{Z}'\right] = \mathbf{Y}_S.$$

Multiply $\mathbf{X}_{\lambda_S}^+$ on both sides. Since $\mathbf{X}_{\lambda_S} \mathbf{X}_{\lambda_S}^+ = \mathbf{I}_d$,

$$\mathbf{W} = \mathbf{Y}_S \mathbf{X}_{\lambda_S}^+ = \mathbf{W}_S.$$

From these two cases, we can also conclude that when $m = d$, such $\mathbf{X}_{\lambda_S} = \mathbf{W}^+ \mathbf{Y}_S + \left(\mathbf{I}_d - \mathbf{W}^+ \mathbf{W}\right) \mathbf{Z}'$ is a sufficient and necessary condition for $\mathbf{W}_S = \mathbf{W}$.

$\square$

Below we give a sufficient condition of $\mathbf{X}_{\lambda_S}$ when $m \geq k$. It will be used in the proof of Theorem 5.3.

**Theorem C.2.** *When $m \geq k$, a sufficient condition for $\mathbf{W}_S = \mathbf{W}$ is $\mathbf{Y}_S$ is rank $k$ and*

$$\mathbf{X}_{\lambda_S} = \mathbf{W}^+ \mathbf{Y}_S + \left(\mathbf{I}_d - \mathbf{W}^+ \mathbf{W}\right) \mathbf{Z}' \left(\mathbf{I}_m - \mathbf{Y}_S^+ \mathbf{Y}_S\right),$$

*where $\mathbf{Z}' \in \mathbb{R}^{d \times m}$ is any matrix of the same size as $\mathbf{X}_{\lambda_S}$.*

*Proof.* When $m \geq k$, for the sufficient condition $\mathbf{X}_{\lambda_S} = \mathbf{W}^+ \mathbf{Y}_S + \left(\mathbf{I}_d - \mathbf{W}^+ \mathbf{W}\right) \mathbf{Z}' \left(\mathbf{I}_m - \mathbf{Y}_S^+ \mathbf{Y}_S\right)$, denote $\mathbf{A} = \mathbf{W}^+ \mathbf{Y}_S$ and $\mathbf{B} = \left(\mathbf{I}_d - \mathbf{W}^+ \mathbf{W}\right) \mathbf{Z}' \left(\mathbf{I}_m - \mathbf{Y}_S^+ \mathbf{Y}_S\right)$. Since $\mathbf{A}^\top \mathbf{B} = 0$ and $\mathbf{A} \mathbf{B}^\top = 0$, by [3], $\left(\mathbf{A} + \mathbf{B}\right)^+ = \mathbf{A}^+ + \mathbf{B}^+$. Therefore

$$
\begin{aligned}
\mathbf{X}_{\lambda_S}^+ &= \left(\mathbf{W}^+ \mathbf{Y}_S\right)^+ + \left[\left(\mathbf{I}_d - \mathbf{W}^+ \mathbf{W}\right) \mathbf{Z}' \left(\mathbf{I}_m - \mathbf{Y}_S^+ \mathbf{Y}_S\right)\right]^+ \\
&= \mathbf{Y}_S^+ \mathbf{W} + \left[\left(\mathbf{I}_d - \mathbf{W}^+ \mathbf{W}\right) \mathbf{Z}' \left(\mathbf{I}_m - \mathbf{Y}_S^+ \mathbf{Y}_S\right)\right]^+
\end{aligned}
$$

where the last equality is because $\mathbf{W}$ and $\mathbf{Y}_S$ are full rank and therefore $\left(\mathbf{W}^+ \mathbf{Y}_S\right)^+ = \mathbf{Y}_S^+ \mathbf{W}$. From this, we have

$$
\begin{aligned}
\mathbf{W}_S &= \mathbf{Y}_S \mathbf{X}_{\lambda_S}^+ \\
&= \mathbf{Y}_S \left(\mathbf{Y}_S^+ \mathbf{W} + \left[\left(\mathbf{I}_d - \mathbf{W}^+ \mathbf{W}\right) \mathbf{Z}' \left(\mathbf{I}_m - \mathbf{Y}_S^+ \mathbf{Y}_S\right)\right]^+\right) \\
&= \mathbf{W} + \mathbf{Y}_S \left[\left(\mathbf{I}_d - \mathbf{W}^+ \mathbf{W}\right) \mathbf{Z}' \left(\mathbf{I}_m - \mathbf{Y}_S^+ \mathbf{Y}_S\right)\right]^+
\end{aligned}
$$

For any matrix $\mathbf{A}$ and $\mathbf{B}$, if $\mathbf{A} \mathbf{B} = 0$ then $\mathbf{B}^+ \mathbf{A}^+ = 0$ [30]. Since $\left(\mathbf{I}_d - \mathbf{W}^+ \mathbf{W}\right) \mathbf{Z}' \left(\mathbf{I}_m - \mathbf{Y}_S^+ \mathbf{Y}_S\right) \mathbf{Y}_S^+ = 0$, $\mathbf{Y}_S \left[\left(\mathbf{I}_d - \mathbf{W}^+ \mathbf{W}\right) \mathbf{Z}' \left(\mathbf{I}_m - \mathbf{Y}_S^+ \mathbf{Y}_S\right)\right]^+ = 0$. Therefore we conclude $\mathbf{W}_S = \mathbf{W}$. Note in this case, we do not require $\mathbf{X}_{\lambda_S}$ to be full rank.

$\square$

### C.3 Finding Realistic Distilled Data

**Corollary 4.1.1.** *Given fixed $\hat{\mathbf{X}}_S, \lambda_S$, and $\mathbf{Y}_S$, the $\mathbf{D}$ that satisfies Theorem 4.1 and minimize $\left\|\mathbf{D} - \hat{\mathbf{X}}_{\lambda_S}^+\right\|_F$ is*

$$\mathbf{D} = \mathbf{Y}_S^+ \mathbf{W} + \left(\mathbf{I}_m - \mathbf{Y}_S^+ \mathbf{Y}_S\right) \left(\hat{\mathbf{X}}_{\lambda_S}^+ - \mathbf{Y}_S^+ \mathbf{W}\right),$$

*where $\hat{\mathbf{X}}_{\lambda_S}^+$ is defined analogous to $\mathbf{X}_{\lambda_S}^+$. Taking $\mathbf{Y}_S = \mathbf{W} \hat{\mathbf{X}}_{\lambda_S}$ can further minimize the distance.*

*Proof.* Given fixed $\hat{\mathbf{X}}_S, \lambda_S$, and $\mathbf{Y}_S$, the linear ridge regression trained on $\hat{\mathbf{X}}_S$ is

$$\mathbf{W}_S = \mathbf{Y}_S \hat{\mathbf{X}}_{\lambda_S}^+ \in \mathbb{R}^{m \times d}$$

By Theorem 4.1, to ensure $\mathbf{W}_S = \mathbf{W}$, we need $\hat{\mathbf{X}}_{\lambda_S}^+$ to be equal to some $\mathbf{D} = \mathbf{Y}_S^+ \mathbf{W} + \left(\mathbf{I}_m - \mathbf{Y}_S^+ \mathbf{Y}_S\right) \mathbf{Z}$, where $\mathbf{Z}$ is a free variable to be determined. Therefore let

$$\mathbf{Y}_S^+ \mathbf{W} + \left(\mathbf{I}_m - \mathbf{Y}_S^+ \mathbf{Y}_S\right) \mathbf{Z} = \hat{\mathbf{X}}_{\lambda_S}^+$$

That is

$$\left(\mathbf{I}_m - \mathbf{Y}_S^+ \mathbf{Y}_S\right) \mathbf{Z} = \hat{\mathbf{X}}_{\lambda_S}^+ - \mathbf{Y}_S^+ \mathbf{W}$$

Since $\left(\mathbf{I}_m - \mathbf{Y}_S^+ \mathbf{Y}_S\right)$ is idempotent and therefore singular, $\mathbf{Z}$ does not have a solution in general (because the system of equations can be inconsistent). The least-squares solution is

$$\mathbf{Z} = \left(\mathbf{I}_m - \mathbf{Y}_S^+ \mathbf{Y}_S\right)^+ \left(\hat{\mathbf{X}}_{\lambda_S}^+ - \mathbf{Y}_S^+ \mathbf{W}\right) = \left(\mathbf{I}_m - \mathbf{Y}_S^+ \mathbf{Y}_S\right) \left(\hat{\mathbf{X}}_{\lambda_S}^+ - \mathbf{Y}_S^+ \mathbf{W}\right)$$

where one can verify that $\left(\mathbf{I}_m - \mathbf{Y}_S^+ \mathbf{Y}_S\right)^+ = \mathbf{I}_m - \mathbf{Y}_S^+ \mathbf{Y}_S$ by SVD. This least-squares solution minimize $\left\| \left(\mathbf{I}_m - \mathbf{Y}_S^+ \mathbf{Y}_S\right) \mathbf{Z} - \hat{\mathbf{X}}_{\lambda_S}^+ + \mathbf{Y}_S^+ \mathbf{W} \right\|_F = \left\| \mathbf{D} - \hat{\mathbf{X}}_{\lambda_S}^+ \right\|_F$. Take such $\mathbf{Z}$ into $\mathbf{D}$, we have

$$\mathbf{D} = \mathbf{Y}_S^+ \mathbf{W} + \left(\mathbf{I}_m - \mathbf{Y}_S^+ \mathbf{Y}_S\right)\left(\mathbf{I}_m - \mathbf{Y}_S^+ \mathbf{Y}_S\right)\left(\hat{\mathbf{X}}_{\lambda_S}^+ - \mathbf{Y}_S^+ \mathbf{W}\right)$$

$$= \mathbf{Y}_S^+ \mathbf{W} + \left(\mathbf{I}_m - \mathbf{Y}_S^+ \mathbf{Y}_S\right)\left(\hat{\mathbf{X}}_{\lambda_S}^+ - \mathbf{Y}_S^+ \mathbf{W}\right)$$

Then we have the difference between $\mathbf{D}$ and $\hat{\mathbf{X}}_{\lambda_S}^+$ is

$$\hat{\mathbf{X}}_{\lambda_S}^+ - \mathbf{D} = \hat{\mathbf{X}}_{\lambda_S}^+ - \mathbf{Y}_S^+ \mathbf{W} - \left(\mathbf{I}_m - \mathbf{Y}_S^+ \mathbf{Y}_S\right)\left(\hat{\mathbf{X}}_{\lambda_S}^+ - \mathbf{Y}_S^+ \mathbf{W}\right)$$

$$= \mathbf{Y}_S^+ \mathbf{Y}_S \left(\hat{\mathbf{X}}_{\lambda_S}^+ - \mathbf{Y}_S^+ \mathbf{W}\right)$$

$$= \mathbf{Y}_S^+ \left(\mathbf{Y}_S \hat{\mathbf{X}}_{\lambda_S}^+ - \mathbf{W}\right)$$

To further minimize the difference, we can let $\mathbf{Y}_S \hat{\mathbf{X}}_{\lambda_S}^+ = \mathbf{W}$. The least square solution is $\mathbf{Y}_S = \mathbf{W} \hat{\mathbf{X}}_{\lambda_S}$.

$\square$

## C.4    Label Distillation

**Theorem 4.2.** *For any fixed $\mathbf{X}_S$,*

1. *when $m < d$, there is no $\mathbf{Y}_S$ can guarantee $\mathbf{W}_S = \mathbf{W}$ in general unless the rows of $\mathbf{W}$ are in the row space of $\mathbf{X}_{\lambda_S}^+$. The least square solution is $\mathbf{Y}_S = \mathbf{W} \mathbf{X}_{\lambda_S}$ and $\|\mathbf{W}_S - \mathbf{W}\| = \left\|\mathbf{W}\left(\mathbf{X}_{\lambda_S} \mathbf{X}_{\lambda_S}^+ - \mathbf{I}_d\right)\right\|$.*

2. *when $m \geq d$, if $\mathbf{X}_S$ is rank $d$, then $\mathbf{Y}_S = \mathbf{W} \mathbf{X}_{\lambda_S}$ is sufficient for $\mathbf{W}_S = \mathbf{W}$.*

*Proof.* When $m < d$, let

$$\mathbf{W}_S = \mathbf{Y}_S \mathbf{X}_{\lambda_S}^+ = \mathbf{W}$$

and solve $\mathbf{Y}_S$. $\mathbf{Y}_S$ does not have a solution in general unless the equations are consistent, i.e. the rows of $\mathbf{W}$ are in the row space of $\mathbf{X}_{\lambda_S}^+$. The least-squares solution is

$$\mathbf{Y}_S = \mathbf{W} \mathbf{X}_{\lambda_S}$$

Therefore we have

$$\mathbf{W}_S = \mathbf{Y}_S \mathbf{X}_{\lambda_S} = \mathbf{W} \mathbf{X}_{\lambda_S} \mathbf{X}_{\lambda_S}^+$$

Then we can bound the difference between $\mathbf{W}_S$ and $\mathbf{W}$,

$$\|\mathbf{W}_S - \mathbf{W}\| = \left\|\mathbf{W}\left(\mathbf{X}_{\lambda_S} \mathbf{X}_{\lambda_S}^+ - \mathbf{I}_d\right)\right\| \leq \|\mathbf{W}\| \left\|\left(\mathbf{X}_{\lambda_S} \mathbf{X}_{\lambda_S}^+ - \mathbf{I}_d\right)\right\|.$$

When $m \geq d$, let

$$\mathbf{W}_S = \mathbf{Y}_S \mathbf{X}_{\lambda_S}^+ = \mathbf{W}$$

and solve $\mathbf{Y}_S$. Since $\mathbf{X}_S$ is rank $d$, then $\mathbf{X}_{\lambda_S}^+ = \mathbf{X}_S^\top \left( \mathbf{X}_S \mathbf{X}_S^\top + \lambda_S \mathbf{I}_d \right)^{-1}$ is rank $d$ and $\mathbf{Y}_S$ has solutions. Take the minimum norm one,

$$\mathbf{Y}_S = \mathbf{W} \mathbf{X}_{\lambda_S}$$

To show the sufficiency, take $\mathbf{Y}_S$ into $\mathbf{W}_S$.

$$\mathbf{W}_S = \mathbf{W} \mathbf{X}_{\lambda_S}^+ \mathbf{X}_{\lambda_S} = \mathbf{W}$$

$\square$

# D   Proofs for Kernel Ridge Regression

**Theorem 5.1.** *When $m < k$, there is no $\phi(\mathbf{X}_S)$ can guarantee $\mathbf{W}_S = \mathbf{W}$ unless the columns of $\mathbf{W}$ are in the range space of $\mathbf{Y}_S$. When $m \geq k$ and $\mathbf{Y}_S$ is rank $k$, let $r = \min(m, p)$ and take $\mathbf{D} = \mathbf{Y}_S^+\mathbf{W} + \left(\mathbf{I}_m - \mathbf{Y}_S^+\mathbf{Y}_S\right)\mathbf{Z}$, where $\mathbf{Z} \in \mathbb{R}^{m \times p}$ is any matrix of the same size as $\phi(\mathbf{X}_S)^\top$. Suppose the reduced SVD of $\mathbf{D}$ is $\mathbf{D} = \mathbf{V}diag(\sigma_1', \ldots, \sigma_r')\mathbf{U}^\top$ with $\sigma_1' \geq \cdots \geq \sigma_r' \geq 0$, the following results hold:*

1. $\lambda_S > 0$: $\mathbf{W}_S = \mathbf{W}$ *if and only if, for any $\mathbf{D}$ defined above, $\lambda_S \leq \frac{1}{4\sigma_1'^2}$ and $\phi(\mathbf{X}_S) = \mathbf{U}diag(\sigma_1, \ldots, \sigma_r)\mathbf{V}^\top$ where* $\sigma_i = \begin{cases} 0, & \text{if } \sigma_i' = 0, \\ \frac{1 \pm \sqrt{1 - 4\lambda_S\sigma_i'^2}}{2\sigma_i'}, & \text{otherwise.} \end{cases}$

2. $\lambda_S = 0$: $\mathbf{W}_S = \mathbf{W}$ *if and only if $\phi(\mathbf{X}_S) = \mathbf{D}^+$ for any $\mathbf{D}$ defined above.*

*Proof.* The proof is same as Theorem 4.1 but just replace $\mathbf{X}_S$ with $\phi(\mathbf{X}_S)$. □

## D.1   Deep Nonlinear Neural Networks

**Theorem 5.2.** *For a deep nonlinear NN defined above with fixed $\phi$, assume $\mathbf{W}^{(2)}, \ldots, \mathbf{W}^{(L)}$ are full rank. Suppose $\lambda_S = 0$ and $\mathbf{Y}_S$ is rank $k$. When $m = k$, there is no distilled data $\mathbf{X}_S$ that can guarantee $\mathbf{W}_S = \mathbf{W}$ in general useless the columns of $\sigma^{-1}\left(\left(\mathbf{W}^{(2)}\right)^{-1}\cdots\left(\mathbf{W}^{(L)}\right)^{-1}\sigma^{-1}\left((\mathbf{Y}_S^+\mathbf{W})^+\right)\right)$ are in the range space of $\mathbf{W}^{(1)}$.*

*Proof.* To get a distilled data $\mathbf{X}_S$ that can guarantee $\mathbf{W}_S = \mathbf{W}$, the sufficient and necessary condition is that

1. $\phi(\mathbf{X}_S)$ need guarantee $\mathbf{W}_S = \mathbf{W}$.

2. There is some $\mathbf{X}_S$ corresponds to such $\phi(\mathbf{X}_S)$. Equivalently $\mathbf{X}_S$ is recoverable from $\phi(\mathbf{X}_S)$.

**1. For the first condition,** when $\lambda_S = 0$, we have shown in Theorem 5.1, $\phi(\mathbf{X}_S)$ has to be

$$\phi(\mathbf{X}_S) = \left(\mathbf{Y}_S^+\mathbf{W} + \left(\mathbf{I}_m - \mathbf{Y}_S^+\mathbf{Y}_S\right)\mathbf{Z}\right)^+ \tag{6}$$

When $m = k$ and $\mathbf{Y}_S$ is rank $k$, this reduce to $\phi(\mathbf{X}_S) = (\mathbf{Y}_S^+\mathbf{W})^+$.

**2. For the second condition,** given $\phi(\mathbf{X}_S) = \sigma\left(\mathbf{W}^{(L)}\cdots\sigma\left(\mathbf{W}^{(1)}\mathbf{X}_S\right)\right)$, solving $\mathbf{X}_S$ is same as solving

$$\mathbf{W}^{(1)}\mathbf{X}_S = \sigma^{-1}\left(\left(\mathbf{W}^{(2)}\right)^{-1}\cdots\left(\mathbf{W}^{(L)}\right)^{-1}\sigma^{-1}\left(\phi(\mathbf{X}_S)\right)\right)$$

When $m = k$ and combined with the first condition, it becomes

$$\mathbf{W}^{(1)}\mathbf{X}_S = \sigma^{-1}\left(\left(\mathbf{W}^{(2)}\right)^{-1}\cdots\left(\mathbf{W}^{(L)}\right)^{-1}\sigma^{-1}\left((\mathbf{Y}_S^+\mathbf{W})^+\right)\right)$$

Since this is an over-determined system of linear equations and RHS is fixed, It does not have a solution in general unless The RHS is in the range space of $\mathbf{W}^{(1)}$.

□

## D.2   Deep Linear Neural Networks

**Theorem 5.3.** *For a deep linear NN defined above with fixed $\phi$, assume $\mathbf{W}^{(2)}, \ldots, \mathbf{W}^{(L)}$ are full rank. Suppose $\lambda_S = 0$ and $\mathbf{Y}_S, \mathbf{W}$ are rank $k$. Denote $\mathbf{H} = \left[\prod_{l=1}^{L}\mathbf{W}^{(l)}\left(\mathbf{W}^{(1)}\right)^+ \quad (\mathbf{W}^+\mathbf{W} - \mathbf{I}_p)\right] \in \mathbb{R}^{p \times 2p}$.*

1. *When $m = k$, there is no distilled data $\mathbf{X}_S$ that can guarantee $\mathbf{W}_S = \mathbf{W}$ in general useless the columns of $\mathbf{W}^+\mathbf{Y}_S$ are in the range space of $\prod_{l=1}^{L}\mathbf{W}^{(l)}$.*

2. When $m > k$, If $\mathbf{H}$ is full rank and its right singular vectors $\mathbf{V_H} \in \mathbb{R}^{2p \times 2p}$'s last $p \times p$ submatrix is full rank, then there exists a $\mathbf{X}_S$ such that $\mathbf{W}_S = \mathbf{W}$.

*Proof.* To get a distilled data $\mathbf{X}_S$ that can guarantee $\mathbf{W}_S = \mathbf{W}$, the sufficient and necessary condition is that

1. $\phi(\mathbf{X}_S)$ need guarantee $\mathbf{W}_S = \mathbf{W}$.

2. There is some $\mathbf{X}_S$ corresponds to such $\phi(\mathbf{X}_S)$. Equivalently $\mathbf{X}_S$ is solvable from $\phi(\mathbf{X}_S)$.

**1. For the first condition,** when $\lambda_S = 0$, we have shown in Theorem 5.1, $\phi(\mathbf{X}_S)$ has to be

$$\phi(\mathbf{X}_S) = \left(\mathbf{Y}_S^+ \mathbf{W} + \left(\mathbf{I}_m - \mathbf{Y}_S^+ \mathbf{Y}_S\right) \mathbf{Z}\right)^+ \tag{7}$$

When $m = k$ and $\mathbf{Y}_S, \mathbf{W}$ are rank $k$, this reduce to $\phi(\mathbf{X}_S) = \mathbf{W}^+ \mathbf{Y}_S$. When $m \geq k$, from Theorem C.2, we know $\phi(\mathbf{X}_S) = \mathbf{W}^+ \mathbf{Y}_S + (\mathbf{I}_p - \mathbf{W}^+ \mathbf{W}) \mathbf{Z}' \left(\mathbf{I}_m - \mathbf{Y}_S^+ \mathbf{Y}_S\right)$ for any $\mathbf{Z}' \in \mathbb{R}^{p \times m}$ is also a sufficient condition for $\mathbf{W}_S = \mathbf{W}$.

**2. For the second condition,** given $\phi(\mathbf{X}_S) = \prod_{l=1}^{L} \mathbf{W}^{(l)} \mathbf{X}_S$, solving $\mathbf{X}_S$ is same as solving

$$\mathbf{W}^{(1)} \mathbf{X}_S = \left(\prod_{l=2}^{L} \mathbf{W}^{(l)}\right)^{-1} \phi(\mathbf{X}_S)$$

This is an over-determined system of linear equations. A necessary and sufficient condition for any solution(s) to exist is that RHS is in the range space of $\mathbf{W}^{(1)}$ or equivalently $\mathbf{X}_S = \left(\mathbf{W}^{(1)}\right)^+ \left(\prod_{l=2}^{L} \mathbf{W}^{(l)}\right)^{-1} \phi(\mathbf{X}_S)$ is a solution. Take this solution into equation,

$$\mathbf{W}^{(1)} \left(\mathbf{W}^{(1)}\right)^+ \left(\prod_{l=2}^{L} \mathbf{W}^{(l)}\right)^{-1} \phi(\mathbf{X}_S) = \left(\prod_{l=2}^{L} \mathbf{W}^{(l)}\right)^{-1} \phi(\mathbf{X}_S)$$

$$\left(\mathbf{I}_p - \mathbf{W}^{(1)} \left(\mathbf{W}^{(1)}\right)^+\right) \left(\prod_{l=2}^{L} \mathbf{W}^{(l)}\right)^{-1} \phi(\mathbf{X}_S) = 0$$

Solve the equation for $\left(\prod_{l=2}^{L} \mathbf{W}^{(l)}\right)^{-1} \phi(\mathbf{X}_S)$, we have

$$\left(\prod_{l=2}^{L} \mathbf{W}^{(l)}\right)^{-1} \phi(\mathbf{X}_S) = \left(\mathbf{I}_p - \left(\mathbf{I}_p - \mathbf{W}^{(1)} \left(\mathbf{W}^{(1)}\right)^+\right)^+ \left(\mathbf{I}_p - \mathbf{W}^{(1)} \left(\mathbf{W}^{(1)}\right)^+\right)\right) \mathbf{Z}_1$$

$$= \left(\mathbf{I}_p - \left(\mathbf{I}_p - \mathbf{W}^{(1)} \left(\mathbf{W}^{(1)}\right)^+\right) \left(\mathbf{I}_p - \mathbf{W}^{(1)} \left(\mathbf{W}^{(1)}\right)^+\right)\right) \mathbf{Z}_1$$

$$= \left(\mathbf{I}_p - \left(\mathbf{I}_p - \mathbf{W}^{(1)} \left(\mathbf{W}^{(1)}\right)^+\right)\right) \mathbf{Z}_1$$

$$= \mathbf{W}^{(1)} \left(\mathbf{W}^{(1)}\right)^+ \mathbf{Z}_1$$

for any $\mathbf{Z}_1 \in \mathbb{R}^{p \times m}$. Therefore to guarantee $\mathbf{X}_S$ is solvable from $\phi(\mathbf{X}_S)$, $\phi(\mathbf{X}_S)$ have to be in the form of

$$\phi(\mathbf{X}_S) = \prod_{l=1}^{L} \mathbf{W}^{(l)} \left(\mathbf{W}^{(1)}\right)^+ \mathbf{Z}_1 \tag{8}$$

In this case, $\mathbf{X}_S = \left(\mathbf{W}^{(1)}\right)^+ \mathbf{Z}_1$. For any $\mathbf{Z}_1$, $\mathbf{W}^{(1)} \left(\mathbf{W}^{(1)}\right)^+$ is a projector that projects $\mathbf{Z}_1$ to the range space of $\mathbf{W}^{(1)}$.

**Combing two conditions** (7) **and** (8)**,** we need to solve

$$\prod_{l=1}^{L} \mathbf{W}^{(l)} \left(\mathbf{W}^{(1)}\right)^+ \mathbf{Z}_1 = \left(\mathbf{Y}_S^+ \mathbf{W} + \left(\mathbf{I}_m - \mathbf{Y}_S^+ \mathbf{Y}_S\right) \mathbf{Z}\right)^+$$

for $\mathbf{Z}$ and $\mathbf{Z}_1$.

1. When $m = k$, it becomes

$$\prod_{l=1}^{L} \mathbf{W}^{(l)} \mathbf{X}_S = \mathbf{W}^{+} \mathbf{Y}_S$$

Since $\prod_{l=1}^{L} \mathbf{W}^{(l)} \in \mathbb{R}^{p \times d}$ and $p > d$, the equation has a solution only when $\mathbf{W}^{+}\mathbf{Y}_S$ is in the range space of $\prod_{l=1}^{L} \mathbf{W}^{(l)}$.

2. When $m > k$, from Theorem C.2, we know $\phi(\mathbf{X}_S) = \mathbf{W}^{+}\mathbf{Y}_S + (\mathbf{I}_p - \mathbf{W}^{+}\mathbf{W}) \mathbf{Z}' (\mathbf{I}_m - \mathbf{Y}_S^{+}\mathbf{Y}_S)$ for any $\mathbf{Z}' \in \mathbb{R}^{p \times m}$ is a sufficient condition for $\mathbf{W}_S = \mathbf{W}$. Therefore we can instead solve

$$\prod_{l=1}^{L} \mathbf{W}^{(l)} \left(\mathbf{W}^{(1)}\right)^{+} \mathbf{Z}_1 = \mathbf{W}^{+}\mathbf{Y}_S + (\mathbf{I}_p - \mathbf{W}^{+}\mathbf{W}) \mathbf{Z}' (\mathbf{I}_m - \mathbf{Y}_S^{+}\mathbf{Y}_S)$$

Combine the variables $\mathbf{Z}'$ and $\mathbf{Z}_1$, we have

$$\left[\prod_{l=1}^{L} \mathbf{W}^{(l)} \left(\mathbf{W}^{(1)}\right)^{+} \quad (\mathbf{W}^{+}\mathbf{W} - \mathbf{I}_p)\right] \begin{bmatrix} \mathbf{Z}_1 \\ \mathbf{Z}' (\mathbf{I}_m - \mathbf{Y}_S^{+}\mathbf{Y}_S) \end{bmatrix} = \mathbf{W}^{+}\mathbf{Y}_S$$

Denote $\mathbf{H} = \left[\prod_{l=1}^{L} \mathbf{W}^{(l)} \left(\mathbf{W}^{(1)}\right)^{+} \quad (\mathbf{W}^{+}\mathbf{W} - \mathbf{I}_p)\right] \in \mathbb{R}^{p \times 2p}$. If $\mathbf{H}$ is full rank (rank $p$), then the solutions are

$$\begin{bmatrix} \mathbf{Z}_1 \\ \mathbf{Z}' (\mathbf{I}_m - \mathbf{Y}_S^{+}\mathbf{Y}_S) \end{bmatrix} = \mathbf{H}^{+}\mathbf{W}^{+}\mathbf{Y}_S + (\mathbf{I}_{2p} - \mathbf{H}^{+}\mathbf{H}) \mathbf{Z}_2$$

where $\mathbf{Z}_2 \in \mathbb{R}^{2p \times m}$ is any matrix. For any RHS, we can find a solution for $\mathbf{Z}_1$. Next, we try to find a solution for $\mathbf{Z}'$ and $\mathbf{Z}_2$ such that the equation is consistent. Suppose the full SVD of $\mathbf{H} = \mathbf{U}\mathbf{\Sigma}\mathbf{V}^{\top}$, where $\mathbf{U} \in \mathbb{R}^{p \times p}, \mathbf{\Sigma} \in \mathbb{R}^{p \times 2p}, \mathbf{V} \in \mathbb{R}^{2p \times 2p}$. Then $\mathbf{I}_{2p} - \mathbf{H}^{+}\mathbf{H} = \mathbf{V}\mathrm{diag}(\underbrace{0,\ldots,0}_{p}, \underbrace{1,\ldots,1}_{p})\mathbf{V}^{\top} = [\mathbf{0} \quad \ldots \quad \mathbf{0} \quad \mathbf{V}_{p+1} \quad \ldots \quad \mathbf{V}_p]\mathbf{V}^{\top}$. Then the equation becomes

$$[\mathbf{0} \quad \ldots \quad \mathbf{0} \quad \mathbf{V}_{p+1} \quad \ldots \quad \mathbf{V}_{2p}]\mathbf{V}^{\top}\mathbf{Z}_2 = \begin{bmatrix} \mathbf{Z}_1 \\ \mathbf{Z}' (\mathbf{I}_m - \mathbf{Y}_S^{+}\mathbf{Y}_S) \end{bmatrix} - \mathbf{H}^{+}\mathbf{W}^{+}\mathbf{Y}_S$$

Denote $\mathbf{B} = \mathbf{V}^{\top}\mathbf{Z}_2 = \begin{bmatrix} \boldsymbol{b}_1^{\top} \\ \vdots \\ \boldsymbol{b}_{2p}^{\top} \end{bmatrix} \in \mathbb{R}^{2p \times m}$. Since $\mathbf{Z}_2$ is solvable from any $\mathbf{B}$, we can instead solve $\mathbf{B}$. Then the equation is

$$[\mathbf{V}_{p+1} \quad \ldots \quad \mathbf{V}_{2p}] \begin{bmatrix} \boldsymbol{b}_{p+1}^{\top} \\ \vdots \\ \boldsymbol{b}_{2p}^{\top} \end{bmatrix} = \begin{bmatrix} \mathbf{Z}_1 \\ \mathbf{Z}' (\mathbf{I}_m - \mathbf{Y}_S^{+}\mathbf{Y}_S) \end{bmatrix} - \mathbf{H}^{+}\mathbf{W}^{+}\mathbf{Y}_S$$

Denote $\mathbf{H}^{+}\mathbf{W}^{+}\mathbf{Y}_S = \begin{bmatrix} \mathbf{C}_1 \\ \mathbf{C}_2 \end{bmatrix}$ where $\mathbf{C}_1, \mathbf{C}_2 \in \mathbb{R}^{p \times m}$ are the first and last $p$ rows. Then the equation can be partitioned into two parts

$$[\mathbf{V}_{p+1} \quad \ldots \quad \mathbf{V}_{2p}]_{1:p} \begin{bmatrix} \boldsymbol{b}_{p+1}^{\top} \\ \vdots \\ \boldsymbol{b}_{2p}^{\top} \end{bmatrix} = \mathbf{Z}_1 - \mathbf{C}_1 \tag{9}$$

$$[\mathbf{V}_{p+1} \quad \ldots \quad \mathbf{V}_{2p}]_{p+1:2p} \begin{bmatrix} \boldsymbol{b}_{p+1}^{\top} \\ \vdots \\ \boldsymbol{b}_{2p}^{\top} \end{bmatrix} = \mathbf{Z}' (\mathbf{I}_m - \mathbf{Y}_S^{+}\mathbf{Y}_S) - \mathbf{C}_2 \tag{10}$$

where $[\mathbf{V}_{p+1} \quad \ldots \quad \mathbf{V}_{2p}]_{1:p}$ denotes its first $p$ rows and $[\mathbf{V}_{p+1} \quad \ldots \quad \mathbf{V}_{2p}]_{p+1:2p}$ denotes its last $p$ rows. For the first equation, there is always a solution for $\mathbf{Z}_1$. So we will mainly

care about if there is a solution for the second equation. For the second equation, when $[\mathbf{V}_{p+1} \quad \cdots \quad \mathbf{V}_{2p}]_{p+1:2p}$ is full rank, then there is always a solution for $\begin{bmatrix} \boldsymbol{b}_{p+1}^\top \\ \vdots \\ \boldsymbol{b}_{2p}^\top \end{bmatrix}$ and therefore solutions for $\mathbf{B}$ and $\mathbf{Z}_2$. The equations do not depend on $\boldsymbol{b}_1, \ldots, \boldsymbol{b}_p$ so they can be anything.

In conclusion, when $[\mathbf{V}_{p+1} \quad \cdots \quad \mathbf{V}_{2p}]_{p+1:2p}$ is full rank, there is a solution for $\mathbf{Z}_1$ and $\mathbf{X}_S$. To construct $\mathbf{X}_S$, take any $\mathbf{Z}'$ and $\phi(\mathbf{X}_S) = \mathbf{W}^+ \mathbf{Y}_S + (\mathbf{I}_p - \mathbf{W}^+ \mathbf{W}) \mathbf{Z}' (\mathbf{I}_m - \mathbf{Y}_S^+ \mathbf{Y}_S)$. Then construct $\mathbf{H} = \left[ \prod_{l=1}^L \mathbf{W}^{(l)} \left( \mathbf{W}^{(1)} \right)^+ \quad (\mathbf{W}^+ \mathbf{W} - \mathbf{I}_p) \right] \in \mathbb{R}^{p \times 2p}$ and its SVD $\mathbf{H} = \mathbf{U} \boldsymbol{\Sigma} \mathbf{V}^\top$. If $[\mathbf{V}_{p+1} \quad \cdots \quad \mathbf{V}_{2p}]_{p+1:2p}$ is full rank, solve (10),

$$\begin{bmatrix} \boldsymbol{b}_{p+1}^\top \\ \vdots \\ \boldsymbol{b}_{2p}^\top \end{bmatrix} = [\mathbf{V}_{p+1} \quad \cdots \quad \mathbf{V}_{2p}]_{p+1:2p}^{-1} \left[ \mathbf{Z}' \left( \mathbf{I}_m - \mathbf{Y}_S^+ \mathbf{Y}_S \right) - \mathbf{C}_2 \right]$$

Then we get $\mathbf{Z}_1$ from (9)

$$\mathbf{Z}_1 = [\mathbf{V}_{p+1} \quad \cdots \quad \mathbf{V}_{2p}]_{1:p} \begin{bmatrix} \boldsymbol{b}_{p+1}^\top \\ \vdots \\ \boldsymbol{b}_{2p}^\top \end{bmatrix} + \mathbf{C}_1$$

Then we can construct $\mathbf{X}_S = \left( \mathbf{W}^{(1)} \right)^+ \mathbf{Z}_1$.

$\square$

## D.3  Additional Trainable Layer

Here we consider whether adding an additional trainable layer to the distilled dataset model will help dataset distillation. Suppose original model is $f(x) = \mathbf{W} \phi(x)$ and distilled dataset model is $f_S(x) = \mathbf{W}_S \mathbf{A} \phi(x)$ where $\mathbf{A} \in \mathbb{R}^{p \times p}$ is the additional trainable layer. In this case, the feature of distilled dataset model is $\mathbf{A} \phi(x)$ instead of $\phi(x)$ and the analytical solution for $\mathbf{W}_S$ becomes $\mathbf{W}_S = \mathbf{Y}_S \left( (\mathbf{A} \phi(\mathbf{X}_S))^\top \mathbf{A} \phi(\mathbf{X}_S) + \lambda_S \mathbf{I}_m \right)^{-1} (\mathbf{A} \phi(\mathbf{X}_S))^\top$. Here the objective becomes $\mathbf{W}_S \mathbf{A} = \mathbf{W}$.

**Theorem D.1.** *When $k \le m < p$ and $\lambda_S > 0$, suppose $\mathbf{Y}_S$ is rank $k$, there exists a distilled dataset $(\mathbf{X}_S, \mathbf{Y}_S)$ can guarantee $\mathbf{W}_S \mathbf{A} = \mathbf{W}$ if below equation has a solution for some $\mathbf{Z} \in \mathbb{R}^{m \times p}$ such that $\phi(\mathbf{X}_S)$ is full rank and some $c > 0$:*

$$\frac{c}{c+1} \phi(\mathbf{X}_S)^+ = \mathbf{Y}_S^+ \mathbf{W} + \left( \mathbf{I}_m - \mathbf{Y}_S^+ \mathbf{Y}_S \right) \mathbf{Z}.$$

*Proof.* What we want now is $\mathbf{W}_S \mathbf{A} = \mathbf{W}$. That is

$$\mathbf{Y}_S \left( (\mathbf{A} \phi(\mathbf{X}_S))^\top \mathbf{A} \phi(\mathbf{X}_S) + \lambda_S \mathbf{I}_m \right)^{-1} (\mathbf{A} \phi(\mathbf{X}_S))^\top \mathbf{A} = \mathbf{W}$$

For a given $\mathbf{Y}_S$, since $k \le m$, the solution of LHS is

$$\left( \phi(\mathbf{X}_S)^\top \mathbf{A}^\top \mathbf{A} \phi(\mathbf{X}_S) + \lambda_S \mathbf{I}_m \right)^{-1} \phi(\mathbf{X}_S)^\top \mathbf{A}^\top \mathbf{A} = \mathbf{Y}_S^+ \mathbf{W} + \left( \mathbf{I}_m - \mathbf{Y}_S^+ \mathbf{Y}_S \right) \mathbf{Z}$$

for any $\mathbf{Z} \in \mathbb{R}^{m \times p}$. Denote RHS as $\mathbf{D} = \mathbf{Y}_S^+ \mathbf{W} + \left( \mathbf{I}_m - \mathbf{Y}_S^+ \mathbf{Y}_S \right) \mathbf{Z}$ and multiply the inverse on both sides,

$$\phi(\mathbf{X}_S)^\top \mathbf{A}^\top \mathbf{A} = \left( \phi(\mathbf{X}_S)^\top \mathbf{A}^\top \mathbf{A} \phi(\mathbf{X}_S) + \lambda_S \mathbf{I}_m \right) \mathbf{D}$$

Arrange the terms,

$$\phi(\mathbf{X}_S)^\top \mathbf{A}^\top \mathbf{A} \left( \mathbf{I}_p - \phi(\mathbf{X}_S) \mathbf{D} \right) = \lambda_S \mathbf{D}$$

If there exists a $\mathbf{D}$ such that $\mathbf{I}_p - \phi(\mathbf{X}_S) \mathbf{D}$ is rank $p$,

$$\phi(\mathbf{X}_S)^\top \mathbf{A}^\top \mathbf{A} = \lambda_S \mathbf{D} \left( \mathbf{I}_p - \phi(\mathbf{X}_S) \mathbf{D} \right)^{-1}$$

Here $\phi(\mathbf{X}_S)^\top \in \mathbb{R}^{m \times p}$. Since $m < p$, there are solutions for $\mathbf{A}^\top \mathbf{A}$. Take the minimum norm one:

$$\mathbf{A}^\top \mathbf{A} = \lambda_S \left( \phi(\mathbf{X}_S)^\top \right)^+ \mathbf{D} \left( \mathbf{I}_p - \phi(\mathbf{X}_S) \mathbf{D} \right)^{-1}$$

Since $\mathbf{A}^\top \mathbf{A}$ is symmetric and positive semidefinite, the RHS also needs to be positive semidefinite. A sufficient condition is that $\mathbf{D} \left( \mathbf{I}_p - \phi(\mathbf{X}_S) \mathbf{D} \right)^{-1} = c\phi(\mathbf{X}_S)^+$ for some constant $c > 0$. Solve it we get $\mathbf{D} = \frac{c}{c+1} \phi(\mathbf{X}_S)^+$. In this case, $\mathbf{I}_p - \phi(\mathbf{X}_S)\mathbf{D} = \mathbf{I}_p - \frac{c}{c+1} \phi(\mathbf{X}_S)\phi(\mathbf{X}_S)^+$ is indeed full rank.

By the definition of $\mathbf{D}$ and $\mathbf{D} = \frac{c}{c+1} \phi(\mathbf{X}_S)^+$, the problem boils down to

$$\frac{c}{c+1} \phi(\mathbf{X}_S)^+ = \mathbf{Y}_S^+ \mathbf{W} + \left( \mathbf{I}_m - \mathbf{Y}_S^+ \mathbf{Y}_S \right) \mathbf{Z}$$

$\square$

Compared with Eq. (7), adding one additional trainable layer only relaxes the original equation with constant scaling and does not help too much.

# E Applications

## E.1 An Implication for KIP-type Algorithms

**Theorem 6.1.** *Suppose $\phi(\mathbf{X})$ is full rank and $\mathbf{W}$ is computed with $\lambda = 0$. Then*

*1. when $n \leq p$, the $\phi(\mathbf{X}_S)$ that can guarantee $\mathbf{W}_S = \mathbf{W}$ in Theorem 5.1 is sufficient for $L(\mathbf{X}_S) = 0$.*

*2. when $n \geq p$, the $\phi(\mathbf{X}_S)$ that can guarantee $\mathbf{W}_S = \mathbf{W}$ in Theorem 5.1 is necessary for $L(\mathbf{X}_S) = 0$.*

*Proof.* The $\phi(\mathbf{X}_S)$ in Theorem 5.1 guarantees $\mathbf{W}_S = \mathbf{W}$. Since $\lambda = 0$ and $\phi(\mathbf{X})$ is full rank, $\mathbf{W} = \mathbf{Y}\phi(\mathbf{X})^+$. Therefore we have

$$\mathbf{W}_S = \mathbf{Y}\phi(\mathbf{X})^+$$

When $n \leq p$, multiply $\phi(\mathbf{X})$ on both sides,

$$\mathbf{W}_S\phi(\mathbf{X}) = \mathbf{Y}$$

which implies $L(\mathbf{X}_S) = \|\mathbf{W}_S\phi(\mathbf{X}) - \mathbf{Y}\|^2 = 0$. This shows that $\phi(\mathbf{X}_S)$ in Theorem 5.1 is sufficient for $L(\mathbf{X}_S) = 0$.

When $n \geq p$, the $L(\mathbf{X}_S) = 0$ means

$$\mathbf{W}_S\phi(\mathbf{X}) = \mathbf{Y}$$

Multiply $\phi(\mathbf{X})^+$ on both sides, we have

$$\mathbf{W}_S = \mathbf{Y}\phi(\mathbf{X})^+ = \mathbf{W}.$$

This implies that $\mathbf{W}_S = \mathbf{W}$ is a necessary condition for $L(\mathbf{X}_S) = 0$. Since the $\phi(\mathbf{X}_S)$ in Theorem 5.1 is sufficient and necessary for $\mathbf{W}_S = \mathbf{W}$. Therefore $\phi(\mathbf{X}_S)$ in Theorem 5.1 is necessary for $L(\mathbf{X}_S) = 0$.

□

## E.2 Privacy Preservation of Dataset Distillation

**Proposition 6.1.** *Suppose $n > k$ and $\mathbf{Y}$ is rank $k$. Given $\lambda_S, \phi$, for a distilled dataset $(\mathbf{X}_S, \mathbf{Y}_S)$ that can guarantee $\mathbf{W}_S = \mathbf{W}$ in Theorem 5.1, we can reconstruct $\mathbf{W}$ from $\phi(\mathbf{X}_S)$. However, given $\mathbf{W}$, there are infinitely many solutions for $\phi(\mathbf{X})$.*

*Proof.* Since $\mathbf{W} = \mathbf{W}_S$, we can reconstruct $\mathbf{W}$ by simply compute $\mathbf{W}_S$.

When $n > k$, since $\mathbf{W} = \mathbf{Y}\phi_\lambda(\mathbf{X})^+$ and $\mathbf{Y}$ is rank $k$, given $\mathbf{W}$, there are infinitely many solutions for $\phi_\lambda(\mathbf{X})^+$,

$$\phi_\lambda(\mathbf{X})^+ = \mathbf{Y}^+\mathbf{W} + \left(\mathbf{I}_n - \mathbf{Y}^+\mathbf{Y}\right)\mathbf{Z}$$

for any $\mathbf{Z} \in \mathbb{R}^{n \times p}$. Using a similar approach as the proof of Theorem 4.1, we can solve $\phi(\mathbf{X})$ by SVD and there are infinitely many solutions for $\phi(\mathbf{X})$. Therefore it is impossible to recover $\phi(\mathbf{X})$ without additional information.

□

**Theorem 6.2.** *Under the same setting of Theorem 4.1, suppose that $\lambda = 0$, all data are bounded $\|\boldsymbol{x}_i\|_2 \leq B$, and the smallest singular value of the original datasets is bounded from below $\sigma_{min}(\mathbf{X}) > \sigma_0$. Suppose $\mathbf{Y}_S$ is independent of $\mathbf{X}$ and unknown to the adversary. Let $[\mathbf{Y}_S^+]_i$ denote its $i$-th row. Let $\epsilon, \delta \in (0, 1)$ and take the elements of $\mathbf{Z} \sim \mathcal{N}(0, \sigma^2)$ with $\sigma \geq \max_{i\in[m]} \frac{2\sqrt{\ln(1.25/\delta)}B\|[\mathbf{Y}_S^+]_i\mathbf{Y}\|_2}{\sigma_0^2\epsilon\|[\mathbf{I}_m - \mathbf{Y}_S^+\mathbf{Y}_S]_i\|_2}$, then each row of $\mathbf{X}_S$ is $(\epsilon, \delta)$-differential private with respect to $\mathbf{X}$.*

*Proof.* We prove that $\mathbf{D} = \mathbf{Y}_S^+\mathbf{W} + \left(\mathbf{I}_m - \mathbf{Y}_S^+\mathbf{Y}_S\right)\mathbf{Z}$ is $(\epsilon, \delta)$-differential private with respect to $\mathbf{X}$ using the Gaussian mechanism [6]. Then it follows that $\mathbf{X}_S$ is $(\epsilon, \delta)$-differential private since the computation from $\mathbf{D}$ to $\mathbf{X}_S$ is deterministic and independent with $\mathbf{X}$.

We first show the sensitivity of $\mathbf{Y}_S^+ \mathbf{W}$ is bounded and then use $\left(\mathbf{I}_m - \mathbf{Y}_S^+ \mathbf{Y}_S\right) \mathbf{Z}$ as random Gaussian to apply the Gaussian mechanism. Without loss of generality, suppose we have two datasets $\mathbf{X} = [\boldsymbol{x}_1, \boldsymbol{x}_2, \ldots, \boldsymbol{x}_n]$, $\mathbf{X}' = [\boldsymbol{x}_1', \boldsymbol{x}_2, \ldots, \boldsymbol{x}_n] \in \mathbb{R}^{d \times n}$ that differ only in the first point and their resulting parameters are $\mathbf{W}$ and $\mathbf{W}'$. Since $\lambda = 0$, $\mathbf{W} = \mathbf{Y}\mathbf{X}^+$. For each row of $\mathbf{Y}_S^+ \mathbf{W}$, we have

$$
\begin{aligned}
\left\| [\mathbf{Y}_S^+]_i \mathbf{W} - [\mathbf{Y}_S^+]_i \mathbf{W}' \right\|_2 &= \left\| [\mathbf{Y}_S^+]_i \mathbf{Y} \left( \mathbf{X}^+ - \mathbf{X}'^+ \right) \right\|_2 \\
&\leq \left\| [\mathbf{Y}_S^+]_i \mathbf{Y} \right\|_2 \left\| \mathbf{X}^+ - \mathbf{X}'^+ \right\|_2 \\
&\leq \left\| [\mathbf{Y}_S^+]_i \mathbf{Y} \right\|_2 \left\| \mathbf{X}^+ - \mathbf{X}'^+ \right\|_F \\
&\leq \left\| [\mathbf{Y}_S^+]_i \mathbf{Y} \right\|_2 \left\| \mathbf{X}^+ \right\|_2 \left\| \mathbf{X}'^+ \right\|_2 \left\| \boldsymbol{x}_1 - \boldsymbol{x}_1' \right\|_F \\
&\leq \left\| [\mathbf{Y}_S^+]_i \mathbf{Y} \right\|_2 \frac{2B}{\sigma_0^2}
\end{aligned}
$$

where the third inequality is due to Theorem 2.2 in Meng and Zheng [23]. Therefore the sensitivity of $[\mathbf{Y}_S^+]_i \mathbf{W}$ is bounded. Suppose the elements of $\mathbf{Z} \sim \mathcal{N}(0, \sigma^2)$, the elements of $\left(\mathbf{I}_m - \mathbf{Y}_S^+ \mathbf{Y}_S\right) \mathbf{Z}$ are also Gaussian. The $i$-th row of $\left(\mathbf{I}_m - \mathbf{Y}_S^+ \mathbf{Y}_S\right) \mathbf{Z}$ is

$$
\left[ \mathbf{I}_m - \mathbf{Y}_S^+ \mathbf{Y}_S \right]_i \mathbf{Z}
$$

whose elements are independent Gaussian with the variance $\left\| \left[ \mathbf{I}_m - \mathbf{Y}_S^+ \mathbf{Y}_S \right]_i \right\|_2^2 \sigma^2$. Therefore, for each row $[\mathbf{Y}_S^+]_i \mathbf{W} + \left[ \mathbf{I}_m - \mathbf{Y}_S^+ \mathbf{Y}_S \right]_i \mathbf{Z}$, we can apply the Gaussian mechanism. Let $\epsilon \in (0, 1)$, by Theorem 3.22 in [6], as long as

$$
\left\| \left[ \mathbf{I}_m - \mathbf{Y}_S^+ \mathbf{Y}_S \right]_i \right\|_2 \sigma \geq \sqrt{\ln(1.25/\delta)} \left\| [\mathbf{Y}_S^+]_i \mathbf{Y} \right\|_2 \frac{2B}{\sigma_0^2 \epsilon}
$$

$$
\Leftrightarrow \sigma \geq \frac{2\sqrt{\ln(1.25/\delta)} B \left\| [\mathbf{Y}_S^+]_i \mathbf{Y} \right\|_2}{\sigma_0^2 \epsilon \left\| \left[ \mathbf{I}_m - \mathbf{Y}_S^+ \mathbf{Y}_S \right]_i \right\|_2}
$$

$[\mathbf{Y}_S^+]_i \mathbf{W} + \left[ \mathbf{I}_m - \mathbf{Y}_S^+ \mathbf{Y}_S \right]_i \mathbf{Z}$ is $(\epsilon, \delta)$-differential private. Take $\sigma$ to be the maximum one so that all the rows are $(\epsilon, \delta)$-differential private.

$$
\sigma \geq \max_{i \in [m]} \frac{2\sqrt{\ln(1.25/\delta)} B \left\| [\mathbf{Y}_S^+]_i \mathbf{Y} \right\|_2}{\sigma_0^2 \epsilon \left\| \left[ \mathbf{I}_m - \mathbf{Y}_S^+ \mathbf{Y}_S \right]_i \right\|_2}.
$$

$\square$

